# Measuring Per-Unit Interpretability at Scale Without Humans

**Roland S. Zimmermann**
MPI-IS, Tübingen AI Center

**David Klindt**
Stanford

**Wieland Brendel**
MPI-IS, Tübingen AI Center

## Abstract

In today's era, whatever we can measure at scale, we can optimize. So far, measuring the interpretability of units in deep neural networks (DNNs) for computer vision still requires direct human evaluation and is not scalable. As a result, the inner workings of DNNs remain a mystery despite the remarkable progress we have seen in their applications. In this work, we introduce the first scalable method to measure the per-unit interpretability in vision DNNs. This method does not require any human evaluations, yet its prediction correlates well with existing human interpretability measurements. We validate its predictive power through an interventional human psychophysics study. We demonstrate the usefulness of this measure by performing previously infeasible experiments: (1) A large-scale interpretability analysis across more than 70 million units from 835 computer vision models, and (2) an extensive analysis of how units transform during training. We find an anti-correlation between a model's downstream classification performance and per-unit interpretability, which is also observable during model training. Furthermore, we see that a layer's location and width influence its interpretability. Online version, code and interactive visualizations available at brendel-group.github.io/mis.

## 1 Introduction

With the arrival of the first non-trivial neural networks, researchers got interested in understanding their inner workings [24, 26]. For one, this can be motivated by scientific curiosity; for another, a better understanding might lead to building more reliable, efficient, or fairer models. While the performance of machine learning models has seen a remarkable improvement over the last few years, our understanding of information processing has progressed more slowly. Nevertheless, understanding how complex models — e.g., language models [7] or vision models [34, 50] — work is still an active and growing field of research, coined *mechanistic interpretability* [33]. A common approach in this field is to divide a network into atomic units, hoping they are easier to comprehend. Here, atomic units might refer to individual neurons or channels of (convolutional) layers [34], or general vectors in feature space [12, 23]. Besides this approach, mechanistic interpretability also includes the detection of neural circuits [8, 12] or analysis of global network properties [29].

The goal of understanding the inner workings of a neural network is inherently human-centric: Irrespective of what tools have been used, in the end, humans should have a better comprehension of the network. However, measuring interpretability through human evaluations is time-consuming and costly due to their reliance on human labor [50]. This results in slower research progress, as validating novel hypotheses takes longer. Removing the need for human labor by automating the interpretability measure can open up multiple high-impact research directions: First, it enables the creation of more interpretable networks by explicitly optimizing for interpretability — after all, what we can measure at scale, we can optimize. Second, it allows more efficient research on explanation methods and might increase our understanding of neural networks. Due to the lack of a reliable automated measure, previous work resorted to limited time-consuming human evaluations, partially producing inconclusive results [e.g., 7, 39], highlighting the urgency of finding an automated measure.

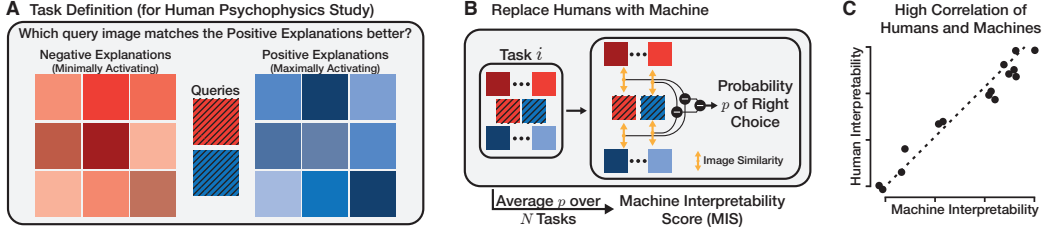

Fig. 1: **Definition of the Machine Interpretability Score. A.** We build on top of the established task definition for quantifying the per-unit interpretability via human psychophysics experiments [6]. The task measures how well participants understand the sensitivity of a unit by asking them to match strongly activating query images to strongly activating *visual* explanations of the unit. Red and blue squares illustrate the unit's minimally and maximally activating images; shaded and solid squares denote natural test images and explanations, respectively. See Fig. 9 for examples. **B.** Crucially, we remove the need for humans and fully automate the evaluation: We pass the explanations and query images through a feature encoder to compute pair-wise image similarities (DreamSim) before using a (hard-coded) binary classifier to solve the underlying task. Finally, the Machine Interpretability Score (MIS) is the average of the predicted probability of the correct choice over $N$ tasks for the same unit. **C.** The MIS proves to be highly correlated with human interpretability ratings and allows fast evaluations of new hypotheses.

The present work is the first to introduce a fully automated interpretability measure (Fig. 1A & B) for vision models: the Machine Interpretability Score (MIS). By leveraging the latest advances in image similarity functions aligned with human perception, we obtain a measure that is strongly predictive of human-perceived interpretability (Fig. 1C). We verify our measure through both correlational and interventional experiments. By removing the need for human labor, we can scale existing evaluations up by multiple orders of magnitude. Finally, this work demonstrates potential workflows and use cases of our MIS.

## 2  Related Work

**Mechanistic Interpretability**   While the overall field of explainable AI (XAI) tries to increase our understanding of neural networks, multiple subbranches with different foci exist [15]. One of these branches, *mechanistic interpretability*, tries to improve our understanding of neural networks by understanding their building blocks [33]. An even more fine-grained branch — per-unit mechanistic interpretability — aims to interpret individual units of vision models [3, 48, 4, 27, 34]. We focus exclusively on this branch of research in the present work. This line of research for artificial neural networks was, arguably, inspired by similar efforts in neuroscience for biological neural networks [20, 2, 37].

Different studies set out to understand the behavior and sensitivity of individual units of vision networks – here, a unit can, e.g., be (the spatial average of) a channel in a convolutional neural network (CNN) or a neuron in a multilayer perceptron (MLP). The level of understanding obtained for a unit is commonly called the *per-unit interpretability*; by averaging over a representative subset of units in the network, one obtains the *per-model interpretability* [50]. With the recent progress in vision-language modeling, a few approaches started using textual descriptions of a unit's behavior [18, 21]. However, the majority still uses visual explanations which are either synthesized by performing activation maximization through gradient ascent [34, 13, 26, 30, 28, 46, 31], or strongly activating dataset examples [34, 6]. With the increasing usage of large language models (LLM), there is also now an increasing interest in mechanistic interpretability of them [e.g., 11, 36, 7].

**Quantifying Interpretability**   Rigorous evaluations, including falsifiable hypothesis testing, are critical for research on interpretability methods [25]. This also encompasses the need for human-centric evaluations [6, 22]. Nevertheless, such human-centric evaluations of interpretability methods are only available in some sub-fields. Specifically for the type of interpretability we are concerned about in this work, i.e., the per-unit interpretability of vision models, two methods for quantifying the helpfulness of explanations to humans were introduced before: Borowski et al. [6] presented a two-alternative-forced-choice (2-AFC) psychophysics task that requires participants to determine

which of two images elicits higher activation of the unit in question, given visual explanations (i.e., images that strongly activate or deactivate the unit, see Fig. 1A) of the unit's behavior. Zimmermann et al. [49] extended this paradigm to quantify how well participants can predict the influence of interventions in the form of occlusions in images. While these studies used their paradigms to evaluate the usefulness of different interpretability methods, Zimmermann et al. [50] leveraged them to compare the interpretability of models. Due to the reliance on human experiments, they could only probe the interpretability of 767 units from nine models. We now automatize this evaluation to scale it up by multiple orders of magnitude to more than 70 million units across 835 models.

**Automating Interpretability Research**    To increase the efficiency of interpretability research and scale it to large modern-day networks, the concept of automated interpretability was proposed in the domain of natural language processing [5]. This approach uses an LLM to generate textual descriptions of the behavior of units in another LLM. Follow-up work by Huang et al. [19], however, pointed out potential problems regarding the correctness of the explanations. Besides automating interpretability research of individual units, there are also efforts for automating the discovery and interpretation of neural circuits and subnetworks [9, 43]. To benchmark future fully automated interpretability tools, acting as independent agents, Schwettmann et al. [41] introduced a synthetic benchmark suite inspired by the behavior of neural networks. In computer vision, there are also efforts to automate interpretability research [18, 50]. Hernandez et al. [18] and Oikarinen and Weng [32] map visual to textual explanations of a unit's behavior using automated tools, hoping to increase the efficiency of evaluations. Zimmermann et al. [50] introduced the ImageNet Mechanistic Interpretability (IMI) dataset, containing per-unit interpretability annotations from humans for 767 units, meant to foster research on automating interpretability evaluations.

## 3    Method

We now introduce our fully automated interpretability measure, Machine Interpretability Score (MIS), visualized in Fig. 1. Borowski et al. [6] proposed a psychophysical experiment for quantifying the per-unit interpretability of vision models, i.e., how well humans can infer the sensitivity of a unit in a vision model from visual explanations. Here, a unit can be a channel in a CNN, commonly averaged over space, a neuron in an MLP, or arbitrary linear combinations of different units. The experiment uses a 2-AFC task design (see Fig. 1A) to measure how well humans understand a unit by probing how well they can predict which of two extremely activating (query) images yields a higher activation, after seeing visual explanations. Specifically, two sets of explanations are displayed: highly and weakly activating images, called positive and negative explanations, respectively. See Appx. A.1 for a more detailed task description. We build on top of this paradigm but replace human participants with machines, resulting in a fully automated interpretability metric that requires no humans.

**Definition of the Machine Interpretability Score**    Let $\mathcal{I}$ denote the space of valid input images for a model. For a specific explanation method and a unit in question, we denote the unit's positive and negative visual explanations as sets of images $\mathcal{E}^+ \subseteq \mathcal{I}$ and $\mathcal{E}^- \subseteq \mathcal{I}$, respectively. Further, let $\mathcal{Q}^+ \subseteq \mathcal{I}$ and $\mathcal{Q}^- \subseteq \mathcal{I}$ be the sets of query images with the most extreme (positive and negative) activations. The task by Borowski et al. [6] can now be expressed as: Given explanations $\mathcal{E}^+$ and $\mathcal{E}^-$ and two queries $\mathbf{q}^+ \in \mathcal{Q}^+$ and $\mathbf{q}^- \in \mathcal{Q}^-$, which of the two queries matches $\mathcal{E}^+$ and which $\mathcal{E}^-$ more closely? An intuitive way to solve this binary decision task is to compare each query with every explanation and match the query images to the sets of explanations based on the images' similarities.

To formalize this, we introduce a perceptual (image) similarity function $f : \mathcal{I} \times \mathcal{I} \rightarrow \mathbb{R}$ computing the scalar similarity of two images [47], and an aggregation function $a : \mathbb{R}^K \rightarrow \mathbb{R}$ reducing a set of $K$ similarities to a single one. This allows us to define the function $s : \mathcal{I} \times \mathcal{I}^K \rightarrow \mathbb{R}$ that quantifies the similarity of a single query image to a set of explanations:

$$s(\mathbf{q}, \mathcal{E}) := a\left(\left\{\, f(\mathbf{q}, \mathbf{e}) \mid \mathbf{e} \in \mathcal{E} \,\right\}\right). \tag{1}$$

To decide whether a single query image is more likely to be the positive one, we can compute whether it is more similar to the positive than the negative explanations. We can compute this now for both the positive and the negative query images and get:

$$\Delta_+(\mathbf{q}^+, \mathcal{E}^+, \mathcal{E}^-) = s(\mathbf{q}^+, \mathcal{E}^+) - s(\mathbf{q}^+, \mathcal{E}^-), \tag{2}$$

$$\Delta_-(\mathbf{q}^-, \mathcal{E}^+, \mathcal{E}^-) = s(\mathbf{q}^-, \mathcal{E}^+) - s(\mathbf{q}^-, \mathcal{E}^-). \tag{3}$$

The classification problem will be solved correctly if the similarity of $\mathbf{q}^+$ to $\mathcal{E}^+$ relative to $\mathcal{E}^-$ is stronger than those of $\mathbf{q}^-$. This means we can define the probability of solving the binary classification problem correctly as

$$p(\mathbf{q}^+, \mathbf{q}^-, \mathcal{E}^+, \mathcal{E}^-) := \sigma\Big(\alpha \cdot \big(\Delta_+(\mathbf{q}^+, \mathcal{E}^+, \mathcal{E}^-) - \Delta_-(\mathbf{q}^-, \mathcal{E}^+, \mathcal{E}^-)\big)\Big), \qquad (4)$$

where $\sigma$ denotes the sigmoid function and $\alpha$ is a free parameter to calibrate the classifier's confidence.

We define the *Machine Interpretability Score* (MIS) as the predicted probability of making the right choice, averaged over $N$ tasks for the same unit. Across these different tasks, the query images $\mathbf{q}^+, \mathbf{q}^-$ vary to cover a wider range of the unit's behavior. If the explanation method used is stochastic, it is advisable to also average over different explanations:

$$\text{MIS} = \frac{1}{N} \sum_i^N p(\mathbf{q}_i^+, \mathbf{q}_i^-, \mathcal{E}_i^+, \mathcal{E}_i^-). \qquad (5)$$

Note that the MIS is not a general property of a unit but depends on the explanation method used. A general score can be defined by aggregating the MIS over multiple explanation methods.

**Choice of Hyperparameters.** We use the current state-of-the-art perceptual similarity, DreamSim [14], as $f$. See Appx. C for a sensitivity study on this choice. DreamSim models the perceptual similarity of two images as the cosine similarity of the images' representations from (multiple) computer vision backbones. These were first pre-trained with, e.g., CLIP-style training [38] and then fine-tuned to match human annotations for image similarities of pairs of images. We use the mean to aggregate the distances between a query image and multiple explanations to a single scalar, i.e., $a(x_1, \ldots, x_K) := 1/K \sum_i^K x_i$. To choose $\alpha$, we use the interpretability annotations of IMI [50]: We optimize $\alpha$ over a randomly chosen subset of just 5% of the annotated units to approximately match the value range of human interpretability scores, resulting in $\alpha = 0.16$. Note that $\alpha$ is, in fact, the only free parameter of our metric, resulting in very low chances of overfitting the metric to the IMI dataset. We use the same strategy as Borowski et al. [6], Zimmermann et al. [49] and Zimmermann et al. [50] for generating new tasks (see Appx. A.2). As they used up to 20 tasks per unit, we average over $N = 20$. See Appx. D for a sensitivity study.

## 4 Results

This section is structured into two parts: First, we validate our Machine Interpretability Score (MIS) by showing that it is well correlated with existing interpretability annotations. Then, we demonstrate what type of experiments become feasible by having access to such an automated interpretability measure. Our experiments use the best-working — according to human judgements [6] — visual explanation method, dataset examples, for computing the MIS. We demonstrate the applicability of our method to other interpretability methods (e.g., feature visualizations) in Appx. E. Note that different explanation methods might require different hyperparameters for computing the MIS. Both query images and explanations are chosen from the training set of ImageNet-2012 [40]. When investigating layers whose feature maps have spatial dimensions, we consider the spatial mean over a channel as one unit [e.g., 6]. We ignore units with constant activations from our analysis as there is no behavior to understand (see Appx. F for details). The code for all experiments is included in the supplementary material and will be publicly released.

### 4.1 Validating the Machine Interpretability Score

We validate our MIS measure by using the interpretability annotations in the IMI dataset [50], which will be referred to as Human Interpretability Scores (HIS). The per-unit annotations are responses to the 2-AFC task described in Sec. 3, averaged over $\approx 30$ participants. IMI contains scores for a subset of units for nine models.[1]

#### 4.1.1 MIS Explains Existing Data

First, we reproduce the main result of Zimmermann et al. [50]: A comparison of nine models in terms of their per-unit interpretability. We plot the HIS and MIS values (averaged over all units in a

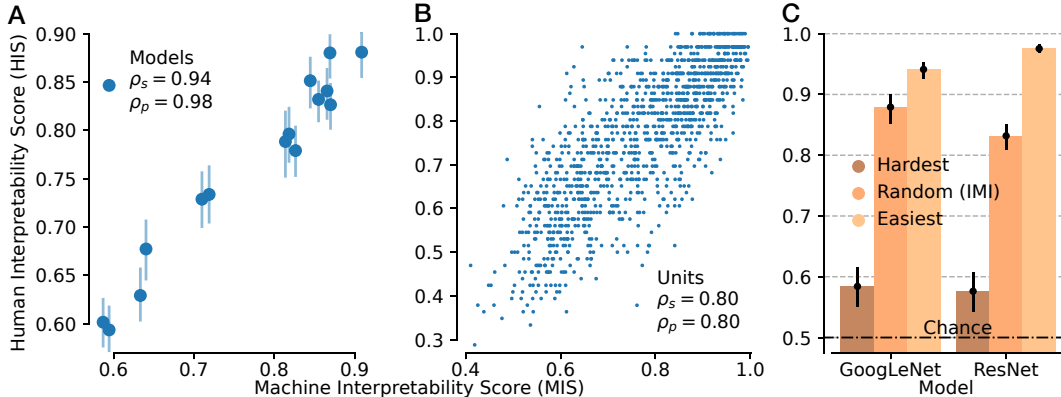

Fig. 2: **Validation of the MIS.** Our proposed Machine Interpretability Score (MIS) explains existing interpretability annotations (Human Interpretability Score, HIS) from IMI [50] well. **(A) MIS Explains Interpretability Model Rankings.** The MIS reproduces the ranking of models presented in IMI while being fully automated and not requiring any human labor, as evident by the strong correlation between MIS and HIS. Similar results are found for the interpretability afforded by another explanation method in Appx. E. **(B) MIS Explains Per-unit Interpretability Annotations.** The MIS also explains individual per-unit interpretability annotations. We show the calculated MIS and the recorded HIS for every unit in IMI and find a high correlation matching the noise ceiling at $\rho = 0.80$ (see Appx. C). **(C) MIS Allows Detection of (Non-) Interpretable Units.** We use the MIS to perform a causal intervention and determine the least (*hardest*) and most (*easiest*) interpretable units in a GoogLeNet and ResNet-50. Using the psychophysics setup of Zimmermann et al. [50], we measure their interpretability and compare them to randomly sampled units. Strikingly, the psychophysics results match the predicted properties: Units with the lowest MIS have significantly lower interpretability than random units, which have significantly lower interpretability than those with the highest MIS. Errorbars denote the 95 % confidence interval.

model) in Fig. 2A and find very strong correlations (Pearson's $r = 0.98$ and Spearman's $r = 0.94$). Reproducing the model ranking is strong evidence for the validity of the metric, as no information about these rankings was explicitly used to create our new measure.

Next, we can zoom in and look at individual units instead of per-model averages. Fig. 2B shows MIS and HIS for all units of IMI. It clearly shows a strong correlation (Pearson's and Spearman's $\rho_s = \rho_p = 0.80$). The interpretability scores in IMI are a (potentially noisy) estimate over a finite number of annotators. We estimate the ceiling performance due to noise (sampling 30 trials from a Bernoulli distribution) to equal Pearson's $\rho_p = 0.82$ (see Appx. C for details). We can conclude that the MIS explains existing interpretability annotations well - both on a per-unit and on a per-model level.

#### 4.1.2 MIS Makes Novel Predictions

While the previous results show a strong relation between MIS and human-perceived interpretability, they are descriptive (correlational). To further test the match between MIS and HIS, we now turn to a causal (interventional) experiment: Instead of predicting the interpretability of units *after* a psychophysics evaluation produced their human scores, we now compute the MIS *before* conducting the psychophysics evaluation. We perform our experiment for two models: GoogLeNet and a ResNet-50. For each model, IMI contains interpretability scores for 96 randomly chosen units. We look at all the units not tested so far and find the 42 units yielding the highest (Easiest, average of 0.99 for both models) and lowest (Hardest, average of 0.63 and 0.59, respectively) MIS, respectively. Then, we use the same setup as Zimmermann et al. [50] and perform a psychophysical evaluation on Amazon Mechanical Turk with 236 participants (Appx. B). We compare the HIS for the random units from the IMI dataset and the two newly recorded groups (easy, hard) of units in Fig. 2C. The results are very clear again: As predicted by the MIS, the HIS is highest for the easiest and lowest for the hardest units. Further, the HIS is close to the *a priori* determined MIS given above. On this newly collected data, we again find a high correlation between MIS and HIS (Pearson's $\rho_p = 0.85$,

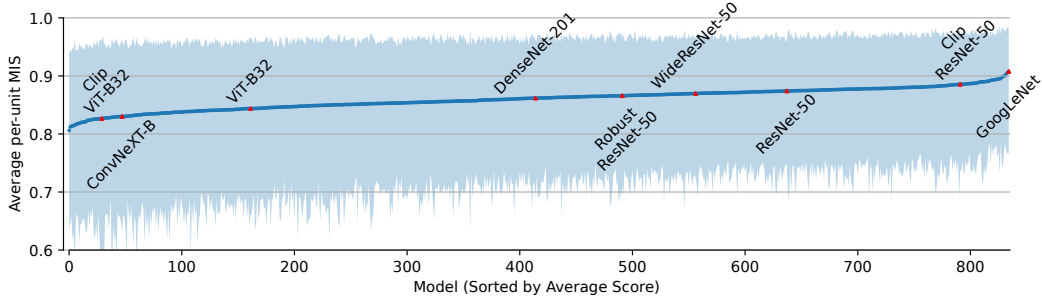

Fig. 3: **Comparison of the Average Per-unit MIS for Models.** We substantially extend the analysis of Zimmermann et al. [50] from a noisy average over a few units for a few models to all units of 835 models. The models are compared regarding their average per-unit interpretability (as judged by MIS); the shaded area depicts the 5th to 95th percentile over units. We see that all models fall into an intermediate performance regime, with stronger changes in interpretability at the tails of the model ranking. Models probed by Zimmermann et al. [50] are highlighted in red.

Spearman's $\rho_s = 0.81$) ). This demonstrates the strong predictive power of the MIS and its ability to be used for formulating novel hypotheses.

## 4.2    Analyzing & Comparing Hundreds of Models

After confirming the validity of the MIS, we now change gears and show use cases for it, i.e., analyses that were truly infeasible before due to the high cost of human evaluations required for measuring the per-unit interpretability. These costs prevented fine-grained analyses. Crucially, our understanding of what influences a unit's interpretability is still fairly limited. For example, it is unclear whether units of specific layer types are more interpretable, or whether a layer's position or width influences its units interpretability. Equipped with the proposed MIS we can now investigate these relations.

### 4.2.1    Comparison of Models

Zimmermann et al. [50] investigated whether model or training design choices influence the interpretability of vision models. Although they invested a considerable amount of money in this investigation ($\geq 12\,000$ USD), they could only compare nine models via a subset of units. We now scale up this line of work by two orders of magnitude and investigate all units of 835 models, almost all of which come from the well-established computer vision library timm [44]. These models differ in architecture and training datasets but were all at least fine-tuned on ImageNet. See Appx. J for a list of models. Putting this scale into perspective, achieving the same scale by scaling up previous human psychophysics experiments would amount to the absurd costs of more than one billion USD. Following previous work we ignore the first and last layers of each model [50].

When sorting the models according to their average MIS (Fig. 3), they span a value range of $\approx 0.80 - 0.91$. The strongest differences across models are present at the tails of the ranking. Note that GoogLeNet is ranked as the most interpretable model, resonating with the community's interest in GoogLeNet as it is widely claimed to be more interpretable. The shaded area denotes the 5th to 95th percentile of the distribution across units. This reveals a strong difference in the variability of units for different models; further, as the upper end of the MIS is similar across models ($\approx 95\,\%$), most of the change in the average score seems to stem from a change in the lower end, with decreasing width of the per-unit distribution for higher model rank. Note that the MIS cannot only be computed for the most extremely activating query images (see Sec. 3) but also for less activating ones. Refer to Fig. 21 for a version of Fig. 3 that uses the 2nd/98th percentile instead of the most extremely activating query images.

To investigate the difference in how the MIS of units is distributed between different models, we select 15 exemplary models and visualize their per-unit MIS distribution in Fig. 4B. Those models were chosen according to the distance between 5th and 95th percentile (five with highest, average, and lowest distance). While models with low and medium variability have unimodal left-skewed distributions, the ones with high variability have a rather bimodal distribution. Note that the distribu-

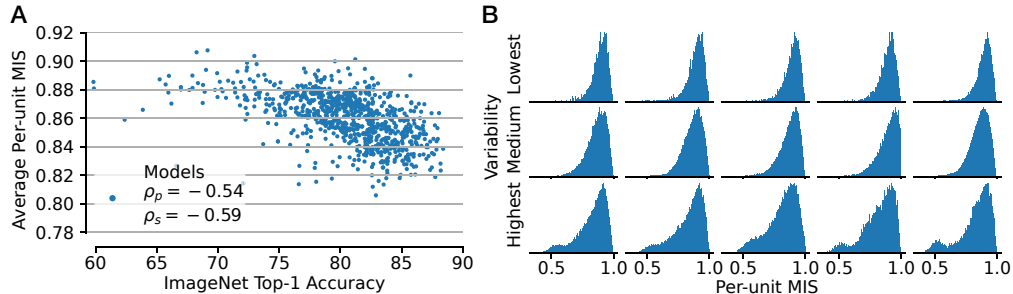

Fig. 4: **(A) Relation Between ImageNet Accuracy and MIS.** The average per-unit MIS of a model is anticorrelated with its ImageNet classification accuracy. Refer to Tab. 2 for a list of the Pareto-optimal models. **(B) Distribution of per-unit MIS.** Distribution of the per-unit MIS for 15 models, chosen based on the size of the error bar in Fig. 3: lowest (top row), medium (middle row), and highest variability (bottom row). While most models have an unimodal distribution, those with high variability have a second mode with lower MIS.

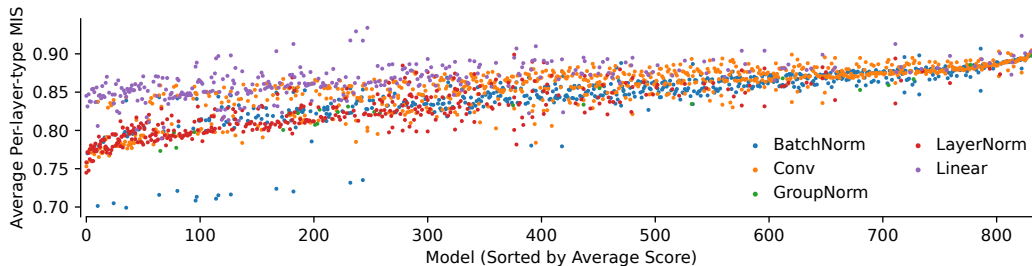

Fig. 5: **Comparison of the Average Per-unit MIS for Different Layer Types and Models.** We show the average interpretability of units from the most common layer types in vision models (BatchNorm, Conv, GroupNorm, LayerNorm, Linear). We follow Zimmermann et al. [50] and restrict our analysis of Vision Transformers to the linear layers in each attention head. While not every layer type is used by every model, we still see some separation between types (see Fig. 18 for significance results): Linear and convolutional layers mostly outperform normalization layers. Models are sorted by average per-unit interpretability, as in Fig. 3.

tion's second, stronger mode has a similar mean and shape to the overall distribution for models with low variability. The first mode is placed at a value range slightly above $0.5$, close to the task's chance level, indicating mostly uninterpretable units. This suggests that a subset of uninterpretable units (see Fig. 28 for examples) can explain most of the models' differences in average MIS. We analyze this further in Fig. 22, where we compare the models in terms of their worst units. We see a similar shape as in Fig. 3, but with a larger value range used, resulting in stronger model differences.

Previous work analyzed a potential correlation between interpretability and downstream classification performance. However, in a limited evaluation, it was found that better classifiers are not necessarily more interpretable [50]. A re-evaluation of this question is performed in Fig. 4A and paints an even darker picture: Here, better performing ImageNet classifiers are less interpretable (Pearson's $r = -0.5$ and Spearman's $r = -0.55$). A similar analysis investigating the influence of a model's input resolution on its interpretability suggests no influence (see Fig. 19).

Besides analyzing the interpretability of models, one can also use the MIS to analyze interpretability tools. Above, we directly looked at the interpretability of a model's activations; however, recent work proposed leveraging sparse auto-encoders (SAE) to first transform a model's activations into a potentially more interpretable basis before analyzing it [e.g., 7]. While their application has been mostly limited to language models (with the exception of [23]), we now apply them to vision models in a first exploratory analysis: In Appx. I, we use the MIS to compare the interpretability of a model's original layer and of two competing SAE variants [39, 7] and find no systematic difference.

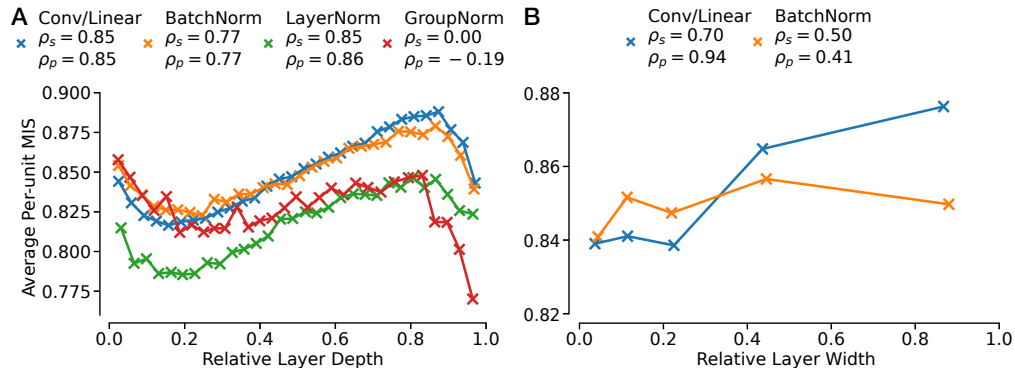

Fig. 6: **(A) Deeper Layers are More Interpretable.** Average MIS per layer as a function of the relative depth of the layer within the network, grouped by layer types. For each type, the values are grouped into 30 bins of equal count based on the relative depth. The crosses depict the bin averages (correlations are calculated for those, too); for a visualization including the bins' variance see Fig. 23. **(B) Wider Layers are More Interpretable.** Average MIS per layer as a function of their relative width, grouped by layer types. The values are grouped into 5 bins. See Fig. 24 for visualizations of how the median, 5th, or 95th percentile of MIS depend on the layer width.

### 4.2.2 Comparison of Layers

Next, we zoom into the results of Fig. 3 and investigate potential differences between layers. First, we are interested in testing whether the layer type is important, e.g., are convolutional more interpretable than normalization or linear layers? In Fig. 5, we sort the models by their average MIS over all layer types but show individual points for each of the five most common types (Conv, Linear, BatchNorm, LayerNorm, and GroupNorm). The number of points per model may vary, as not all models contain layers of all types. The figure shows a benefit of Conv over BatchNorm layers, which themselves are better than LayerNorm layers. Linear layers, if present, outperform both Batch- and LayerNorm as well as Conv layers. While the differences are small, they are statistically significant due to the large number of scores collected (see Fig. 18).

Second, we analyze whether the location of a layer inside a model plays a role, e.g., are earlier layers more interpretable than later ones? The average per-unit MIS (for each layer type) is shown in Fig. 6A as a function of the relative depth of the layer. A value of zero corresponds to the first and a value of one to the last layer analyzed. The scores are averaged in bins of equal count defined by the relative layer depth to enhance readability. The resulting curves all follow a similar pattern: They start high, decrease in the first fifth, then increase steadily until they drop in the last tenth again, resulting in an almost sinusoidal shape.

Third, it is interesting to probe the influence of the width of layers on their average interpretability. Based on the superposition hypothesis [12, 35, 1, 16], one might expect wider layers to be more interpretable as features do not have to form in superposition (i.e., as *polysemantic* units) but can arise in a disentangled form (i.e., as *monosemantic* units). Fig. 6B shows the relation between MIS and relative layer width. We use the relative rather than the absolute width to reduce the influence of the overall model and show the results of models with different architectures on the same axis. Note that, nevertheless, there might be other confounding factors correlated with the width, e.g., the layer depth. While we only see a weak correlation for BatchNorm layers, we find a stronger one for Conv/Linear layers. It is unclear what causes this difference in behavior. However, we see this as a hint that one way to increase a model's interpretability is to increase the width (and not the number) of layers.

### 4.3 How Does the MIS Change During Training?

In the last set of experiments, we demonstrate how the MIS can be used to analyze models in a fine-grained way and obtain insights into their training dynamics. For this, we train a ResNet-50 on ImageNet-2012, following the training recipe A3 of Wightman et al. [45], for 100 epochs.

Fig. 7 shows how the average per-unit MIS (left) changes during the training. Notably, the initial MIS (of the untrained network) is already above chance level. Visual explanations (see supplementary

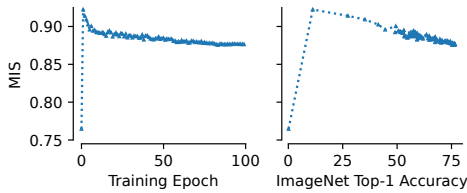

Figure 7: **Interpretability During Training.** For a ResNet-50 trained for 100 epochs on ImageNet, we track the MIS and accuracy after every epoch (epoch 0 refers to initialization). While the MIS improves drastically in the first epoch, it decays during the rest of the training (left). This results in an antiproportional relation between MIS and accuracy (right).

material) indicate a high color dependence of this network's units. However, during the first epoch, the MIS still increases drastically to values around 0.93, before it decays over the rest of the training. This indicates non-trivial dynamics of feature learning, which we analyze in Fig. 8. When showing the MIS as a function of ImageNet accuracy during training (right), a strong anticorrelation (ignoring the first points) becomes evident. This aligns with the anticorrelation shown in Fig. 4A. While we do not have a definite answer for why this is happening, we hypothesize the following: This could be a sign of learning dynamics and the order in which features are learned. After initialization, the network can improve the fastest by learning very simple feature detectors (e.g., colors, simple geometric shapes), as those are weakly correlated with certain classes (e.g., blue colors increase the chance of seeing a fish). Those features are easy for humans to understand. Throughout the training, these feature detectors are replaced with more complex ones that are harder to decode. Fig. 25 the least/most activating dataset examples for units with a strong MIS drop between the second and last training epoch, matching our hypothesis. To better understand the dynamics through the training

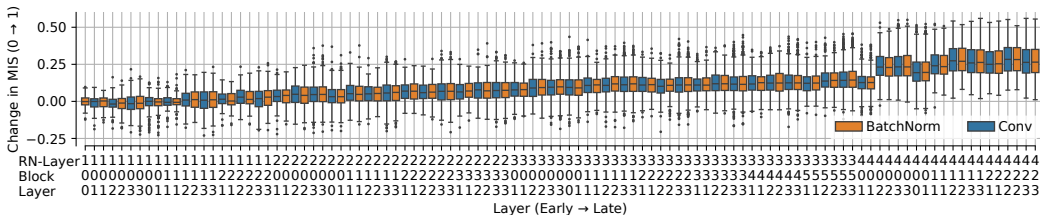

Fig. 8: **Change of Interpretability per Layer During Training.** To better understand the peak in interpretability after the first training epoch found in Fig. 7, we display the change in MIS during the first epoch, averaged over each layer. Layers are sorted by depth from left to right, and different colors encode different layer types. The change in interpretability appears moderately correlated with a layer's depth, such that deeper layers improve the strongest, whereas early layers show no improvement. For an extended visualization covering the full training, see Fig. 20.

— most importantly during the first epoch — we zoom in to find out which units cause this strong change in MIS. Fig. 8 shows the change in MIS during the first epoch for each layer separately (ordered by their depth within the network). We detect a trend of later layers improving more strongly than earlier ones: The change in MIS is heavily driven by the later layers in the network, whose MIS increases strongly while early laters show no improvement at first. In general, we do not see a difference between Conv and BatchNorm layers.

## 5 Conclusion

This paper presented the first fully automated intepretability metric for vision models: the machine interpretability score (MIS). We verified its alignment to human interpretability score (HIS) through both correlational and interventional experiments. We expect our MIS to enable experiments previously considered infeasible due to the costly reliance on human evaluations. To stress this, we demonstrated the metric's usefulness for formulating and testing new hypotheses about a network's behavior through a series of experiments: Based on the largest comparison of vision models in terms of their per-unit interpretability so far, we investigated potential influences on their interpretability, such as layer depth and width. Most importantly, we find an anticorrelation between a model's downstream performance and its per-unit interpretability. Further, we performed the first detailed analysis of how the interpretability changes during training.

While this paper considerably advances the state of interpretability evaluations, there are some open questions and potential future research directions. Most importantly, the performance of our MIS on a per-unit level is close to the noise ceiling determined by the limited number of human interpretability annotations available. This means that future changes in the MIS measure (e.g., based on other image perceptual similarities) might require additional human labels to determine the significance of performance improvements. Additional human labels could also be leveraged to improve the MIS by following Fu et al. [14] to fine-tune the image similarity directly on human judgments. In another direction, using vision language models for computing the MIS could be interesting as this might, in addition to a numerical score, also provide a textual description of a unit's sensitivity [18]. Finding a differentiable approximation of the MIS will be valuable for explicitly training models to be interpretable [50]. Note that while this paper looked at the interpretability of channels and neurons, it can also be used to analyze arbitrary directions in activation space. Thus, we expect the MIS to also be valuable for researchers generally looking for more interpretable representations of (artificial) neural activations [e.g., 17]. Finally, exploring whether this concept of interpretability quantification can be expanded to LLMs is an exciting direction.

**Author Contributions**

RSZ led the project, which DK initiated. DK proposed using perceptual similarity functions to build an interoperability metric. RSZ and WB conceived the final formulation of the metric. RSZ conducted all the experiments with suggestions from WB and feedback from DK. RSZ executed the data analysis, except for the estimation of the noise ceiling conducted by DK. RSZ created all the figures in the paper and wrote the manuscript with suggestions from DK and WB.

**Acknowledgments**

This work was supported by the German Federal Ministry of Education and Research (BMBF): Tübingen AI Center, FKZ: 01IS18039A. WB acknowledges financial support via an Emmy Noether Grant funded by the German Research Foundation (DFG) under grant no. BR 6382/1-1 and via the Open Philantropy Foundation funded by the Good Ventures Foundation. WB is a member of the Machine Learning Cluster of Excellence, EXC number 2064/1 – Project number 390727645. This research utilized compute resources at the Tübingen Machine Learning Cloud, DFG FKZ INST 37/1057-1 FUGG. The authors thank the International Max Planck Research School for Intelligent Systems (IMPRS-IS) for supporting RSZ.

## Footnotes

[1]Two models were tested in multiple settings, resulting in 14 distinct experimental conditions to compare.

# References

[1] Sanjeev Arora, Yuanzhi Li, Yingyu Liang, Tengyu Ma, and Andrej Risteski. Linear Algebraic Structure of Word Senses, with Applications to Polysemy, December 2018. Cited on page 8.

[2] Horace Barlow. Single units and sensation: A neuron doctrine for perceptual psychology? *Perception*, 1:371–94, 02 1972. doi: 10.1068/p010371. Cited on page 2.

[3] David Bau, Bolei Zhou, Aditya Khosla, Aude Oliva, and Antonio Torralba. Network dissection: Quantifying interpretability of deep visual representations. In *Proceedings of the IEEE Conference on Computer Vision and Pattern Recognition (CVPR)*, July 2017. Cited on page 2.

[4] David Bau, Jun-Yan Zhu, Hendrik Strobelt, Agata Lapedriza, Bolei Zhou, and Antonio Torralba. Understanding the role of individual units in a deep neural network. *Proceedings of the National Academy of Sciences*, 117(48):30071–30078, September 2020. doi: 10.1073/pnas.1907375117. URL https://doi.org/10.1073/pnas.1907375117. Cited on page 2.

[5] Steven Bills, Nick Cammarata, Dan Mossing, Henk Tillman, Leo Gao, Gabriel Goh, Ilya Sutskever, Jan Leike, Jeff Wu, and William Saunders. Language models can explain neurons in language models. https://openaipublic.blob.core.windows.net/neuron-explainer/paper/index.html, 2023. Cited on page 3.

[6] Judy Borowski, Roland S. Zimmermann, Judith Schepers, Robert Geirhos, Thomas S. A. Wallis, Matthias Bethge, and Wieland Brendel. Exemplary natural images explain cnn activations better than state-of-the-art feature visualization. In *Ninth International Conference on Learning Representations (ICLR 2021)*, 2021. Cited on pages 2, 3, 4, 15, 16, and 17.

[7] Trenton Bricken, Adly Templeton, Joshua Batson, Brian Chen, Adam Jermyn, Tom Conerly, Nick Turner, Cem Anil, Carson Denison, Amanda Askell, Robert Lasenby, Yifan Wu, Shauna Kravec, Nicholas Schiefer, Tim Maxwell, Nicholas Joseph, Zac Hatfield-Dodds, Alex Tamkin, Karina Nguyen, Brayden McLean, Josiah E Burke, Tristan Hume, Shan Carter, Tom Henighan, and Christopher Olah. Towards monosemanticity: Decomposing language models with dictionary learning. *Transformer Circuits Thread*, 2023. https://transformer-circuits.pub/2023/monosemantic-features/index.html. Cited on pages 1, 2, 7, 20, and 21.

[8] Nick Cammarata, Shan Carter, Gabriel Goh, Chris Olah, Michael Petrov, Ludwig Schubert, Chelsea Voss, Ben Egan, and Swee Kiat Lim. Thread: Circuits. *Distill*, 2020. doi: 10.23915/distill.00024. https://distill.pub/2020/circuits. Cited on page 1.

[9] Arthur Conmy, Augustine N. Mavor-Parker, Aengus Lynch, Stefan Heimersheim, and Adrià Garriga-Alonso. Towards Automated Circuit Discovery for Mechanistic Interpretability, October 2023. Cited on page 3.

[10] Keyan Ding, Kede Ma, Shiqi Wang, and Eero P. Simoncelli. Image Quality Assessment: Unifying Structure and Texture Similarity. *IEEE Transactions on Pattern Analysis and Machine Intelligence*, 44(5):2567–2581, May 2022. ISSN 1939-3539. doi: 10.1109/TPAMI.2020. 3045810. Cited on page 17.

[11] Nelson Elhage, Neel Nanda, Catherine Olsson, Tom Henighan, Nicholas Joseph, Ben Mann, Amanda Askell, Yuntao Bai, Anna Chen, Tom Conerly, Nova DasSarma, Dawn Drain, Deep Ganguli, Zac Hatfield-Dodds, Danny Hernandez, Andy Jones, Jackson Kernion, Liane Lovitt, Kamal Ndousse, Dario Amodei, Tom Brown, Jack Clark, Jared Kaplan, Sam McCandlish, and Chris Olah. A mathematical framework for transformer circuits. *Transformer Circuits Thread*, 2021. https://transformer-circuits.pub/2021/framework/index.html. Cited on page 2.

[12] Nelson Elhage, Tristan Hume, Catherine Olsson, Nicholas Schiefer, Tom Henighan, Shauna Kravec, Zac Hatfield-Dodds, Robert Lasenby, Dawn Drain, Carol Chen, Roger Grosse, Sam McCandlish, Jared Kaplan, Dario Amodei, Martin Wattenberg, and Christopher Olah. Toy models of superposition. *Transformer Circuits Thread*, 2022. Cited on pages 1 and 8.

[13] Dumitru Erhan, Y. Bengio, Aaron Courville, and Pascal Vincent. Visualizing higher-layer features of a deep network. *Technical Report, Univeristé de Montréal*, 01 2009. Cited on page 2.

[14] Stephanie Fu, Netanel Tamir, Shobhita Sundaram, Lucy Chai, Richard Zhang, Tali Dekel, and Phillip Isola. DreamSim: Learning New Dimensions of Human Visual Similarity using Synthetic Data, December 2023. Cited on pages 4, 10, and 17.

[15] Leilani H. Gilpin, David Bau, Ben Z. Yuan, Ayesha Bajwa, Michael A. Specter, and Lalana Kagal. Explaining explanations: An overview of interpretability of machine learning. *2018 IEEE 5th International Conference on Data Science and Advanced Analytics (DSAA)*, pages 80–89, 2018. Cited on page 2.

[16] Gabriel Goh. Decoding the Thought Vector. https://gabgoh.github.io/ThoughtVectors/, 2016. Cited on page 8.

[17] Mara Graziani, Laura O'Mahony, An-phi Nguyen, Henning Müller, and Vincent Andrearczyk. Uncovering unique concept vectors through latent space decomposition. *Transactions on Machine Learning Research*, 2023. Cited on page 10.

[18] Evan Hernandez, Sarah Schwettmann, David Bau, Teona Bagashvili, Antonio Torralba, and Jacob Andreas. Natural Language Descriptions of Deep Visual Features, April 2022. Cited on pages 2, 3, and 10.

[19] Jing Huang, Atticus Geiger, Karel D'Oosterlinck, Zhengxuan Wu, and Christopher Potts. Rigorously assessing natural language explanations of neurons. *arXiv preprint arXiv:2309.10312*, 2023. Cited on page 3.

[20] D H Hubel and T N Wiesel. Receptive fields, binocular interaction and functional architecture in the cat's visual cortex. *J. Physiol.*, 160(1):106–154, January 1962. Cited on page 2.

[21] Neha Kalibhat, Shweta Bhardwaj, C Bayan Bruss, Hamed Firooz, Maziar Sanjabi, and Soheil Feizi. Identifying interpretable subspaces in image representations. In *International Conference on Machine Learning*, pages 15623–15638. PMLR, 2023. Cited on page 2.

[22] Sunnie S. Y. Kim, Nicole Meister, Vikram V. Ramaswamy, Ruth Fong, and Olga Russakovsky. HIVE: Evaluating the human interpretability of visual explanations. In *European Conference on Computer Vision (ECCV)*, 2022. Cited on page 2.

[23] David Klindt, Sophia Sanborn, Francisco Acosta, Frédéric Poitevin, and Nina Miolane. Identifying interpretable visual features in artificial and biological neural systems. *arXiv preprint arXiv:2310.11431*, 2023. Cited on pages 1 and 7.

[24] Alex Krizhevsky, Ilya Sutskever, and Geoffrey E. Hinton. ImageNet Classification with Deep Convolutional Neural Networks. In Peter L. Bartlett, Fernando C. N. Pereira, Christopher J. C. Burges, Léon Bottou, and Kilian Q. Weinberger, editors, *Advances in Neural Information Processing Systems 25: 26th Annual Conference on Neural Information Processing Systems 2012. Proceedings of a Meeting Held December 3-6, 2012, Lake Tahoe, Nevada, United States*, pages 1106–1114, 2012. Cited on pages 1 and 17.

[25] Matthew L. Leavitt and Ari S. Morcos. Towards falsifiable interpretability research. *CoRR*, abs/2010.12016, 2020. URL https://arxiv.org/abs/2010.12016. Cited on page 2.

[26] A. Mahendran and A. Vedaldi. Understanding deep image representations by inverting them. In *2015 IEEE Conference on Computer Vision and Pattern Recognition (CVPR)*, pages 5188–5196, Los Alamitos, CA, USA, jun 2015. IEEE Computer Society. doi: 10.1109/CVPR.2015.7299155. URL https://doi.ieeecomputersociety.org/10.1109/CVPR.2015.7299155. Cited on pages 1 and 2.

[27] Ari S. Morcos, David G.T. Barrett, Neil C. Rabinowitz, and Matthew Botvinick. On the importance of single directions for generalization. In *International Conference on Learning Representations*, 2018. URL https://openreview.net/forum?id=r1iuQjxCZ. Cited on page 2.

[28] Alexander Mordvintsev, Chris Olah, and Mike Tyka. Inceptionism: Going deeper into neural networks, 2015. URL https://ai.googleblog.com/2015/06/inceptionism-going-deeper-into-neural.html. Cited on page 2.

[29] Neel Nanda, Lawrence Chan, Tom Lieberum, Jess Smith, and Jacob Steinhardt. Progress measures for grokking via mechanistic interpretability, 2023. Cited on page 1.

[30] Anh Nguyen, Jason Yosinski, and Jeff Clune. Deep neural networks are easily fooled: High confidence predictions for unrecognizable images. *IEEE Conference on Computer Vision and Pattern Recognition (CVPR)*, 12 2014. Cited on page 2.

[31] Anh Nguyen, Jeff Clune, Yoshua Bengio, Alexey Dosovitskiy, and Jason Yosinski. Plug & play generative networks: Conditional iterative generation of images in latent space. In *Proceedings of the IEEE Conference on Computer Vision and Pattern Recognition*. IEEE, 2017. Cited on page 2.

[32] Tuomas Oikarinen and Tsui-Wei Weng. Clip-dissect: Automatic description of neuron representations in deep vision networks. In *The Eleventh International Conference on Learning Representations*, 2022. Cited on page 3.

[33] Chris Olah. Mechanistic interpretability, variables, and the importance of interpretable bases, 2022. URL `https://transformer-circuits.pub/2022/mech-interp-essay/index.html`. Cited on pages 1 and 2.

[34] Chris Olah, Alexander Mordvintsev, and Ludwig Schubert. Feature visualization. *Distill*, 2017. doi: 10.23915/distill.00007. https://distill.pub/2017/feature-visualization. Cited on pages 1 and 2.

[35] Chris Olah, Nick Cammarata, Ludwig Schubert, Gabriel Goh, Michael Petrov, and Shan Carter. Zoom in: An introduction to circuits. *Distill*, 2020. doi: 10.23915/distill.00024.001. https://distill.pub/2020/circuits/zoom-in. Cited on page 8.

[36] Catherine Olsson, Nelson Elhage, Neel Nanda, Nicholas Joseph, Nova DasSarma, Tom Henighan, Ben Mann, Amanda Askell, Yuntao Bai, Anna Chen, Tom Conerly, Dawn Drain, Deep Ganguli, Zac Hatfield-Dodds, Danny Hernandez, Scott Johnston, Andy Jones, Jackson Kernion, Liane Lovitt, Kamal Ndousse, Dario Amodei, Tom Brown, Jack Clark, Jared Kaplan, Sam McCandlish, and Chris Olah. In-context learning and induction heads. *Transformer Circuits Thread*, 2022. https://transformer-circuits.pub/2022/in-context-learning-and-induction-heads/index.html. Cited on page 2.

[37] R Quian Quiroga, Leila Reddy, Gabriel Kreiman, Christof Koch, and Itzhak Fried. Invariant visual representation by single neurons in the human brain. *Nature*, 435(7045):1102–1107, 2005. Cited on page 2.

[38] Alec Radford, Jong Wook Kim, Chris Hallacy, Aditya Ramesh, Gabriel Goh, Sandhini Agarwal, Girish Sastry, Amanda Askell, Pamela Mishkin, Jack Clark, Gretchen Krueger, and Ilya Sutskever. Learning Transferable Visual Models From Natural Language Supervision, February 2021. Cited on page 4.

[39] Senthooran Rajamanoharan, Arthur Conmy, Lewis Smith, Tom Lieberum, Vikrant Varma, János Kramár, Rohin Shah, and Neel Nanda. Improving Dictionary Learning with Gated Sparse Autoencoders, April 2024. Cited on pages 1, 7, 20, and 21.

[40] Olga Russakovsky, Jia Deng, Hao Su, Jonathan Krause, Sanjeev Satheesh, Sean Ma, Zhiheng Huang, Andrej Karpathy, Aditya Khosla, Michael Bernstein, Alexander C. Berg, and Li Fei-Fei. ImageNet Large Scale Visual Recognition Challenge. *International Journal of Computer Vision (IJCV)*, 115(3):211–252, 2015. doi: 10.1007/s11263-015-0816-y. Cited on pages 4 and 34.

[41] Sarah Schwettmann, Tamar Rott Shaham, Joanna Materzynska, Neil Chowdhury, Shuang Li, Jacob Andreas, David Bau, and Antonio Torralba. FIND: A Function Description Benchmark for Evaluating Interpretability Methods, December 2023. Cited on page 3.

[42] Karen Simonyan and Andrew Zisserman. Very Deep Convolutional Networks for Large-Scale Image Recognition. In Yoshua Bengio and Yann LeCun, editors, *3rd International Conference on Learning Representations, ICLR 2015, San Diego, CA, USA, May 7-9, 2015, Conference Track Proceedings*, 2015. Cited on page 17.

[43] Aaquib Syed, Can Rager, and Arthur Conmy. Attribution Patching Outperforms Automated Circuit Discovery, November 2023. Cited on page 3.

[44] Ross Wightman. Pytorch image models. `https://github.com/rwightman/pytorch-image-models`, 2019. Cited on pages 6 and 21.

[45] Ross Wightman, Hugo Touvron, and Hervé Jégou. Resnet strikes back: An improved training procedure in timm. *arXiv preprint arXiv:2110.00476*, 2021. Cited on page 8.

[46] Jason Yosinski, Jeff Clune, Anh Nguyen, Thomas Fuchs, and Hod Lipson. Understanding neural networks through deep visualization. In *Deep Learning Workshop, International Conference on Machine Learning (ICML)*, 2015. Cited on page 2.

[47] Richard Zhang, Phillip Isola, Alexei A. Efros, Eli Shechtman, and Oliver Wang. The Unreasonable Effectiveness of Deep Features as a Perceptual Metric. In *2018 IEEE Conference on Computer Vision and Pattern Recognition, CVPR 2018, Salt Lake City, UT, USA, June 18-22, 2018*, pages 586–595. Computer Vision Foundation / IEEE Computer Society, 2018. doi: 10.1109/CVPR.2018.00068. Cited on pages 3 and 17.

[48] Bolei Zhou, Yiyou Sun, David Bau, and Antonio Torralba. Revisiting the importance of individual units in cnns via ablation. *CoRR*, abs/1806.02891, 2018. URL `http://arxiv.org/abs/1806.02891`. Cited on page 2.

[49] Roland S. Zimmermann, Judy Borowski, Robert Geirhos, Matthias Bethge, Thomas Wallis, and Wieland Brendel. How well do feature visualizations support causal understanding of cnn activations? In M. Ranzato, A. Beygelzimer, Y. Dauphin, P.S. Liang, and J. Wortman Vaughan, editors, *Advances in Neural Information Processing Systems*, volume 34, pages 11730–11744. Curran Associates, Inc., 2021. URL `https://proceedings.neurips.cc/paper_files/paper/2021/file/618faa1728eb2ef6e3733645273ab145-Paper.pdf`. Cited on pages 3, 4, 15, and 16.

[50] Roland S. Zimmermann, Thomas Klein, and Wieland Brendel. Scale alone does not improve mechanistic interpretability in vision models. In *Thirty-seventh Conference on Neural Information Processing Systems*, 2023. URL `https://openreview.net/forum?id=OZ7aImD4uQ`. Cited on pages 1, 2, 3, 4, 5, 6, 7, 10, 15, 16, 21, and 34.

# A Description of the 2-AFC Task

## A.1 Task Design

Our proposed MIS builds on the 2-AFC task designed by Borowski et al. [6] to conduct human psychophysics experiments. An example of such a task is given in Fig. 9.

This task aims to probe how well (human) participants can detect the sensitivity of a unit of a neural network based on visual explanations of it. Understanding the unit's sensitivity should allow participants to distinguish between a stimulus eliciting high from one yielding low activation. Therefore, the task shows the participants two such images, called query images, and asks them to pick the image eliciting higher activation. To solve the task, participants also see two sets of visual explanations: Positive explanations describe the patterns the unit activates strongly for, while negative activations show patterns the unit weakly responds to. For solving this task, there are two potential strategies: Participants can either recognize a common pattern of the positive explanations in one of the query images, making this the correct choice. Or they detect a common pattern of the negative explanations in a query image, making the other one the right choice. See Borowski et al. [6], Zimmermann et al. [49] or Zimmermann et al. [50] for alternative descriptions and visualizations of the task.

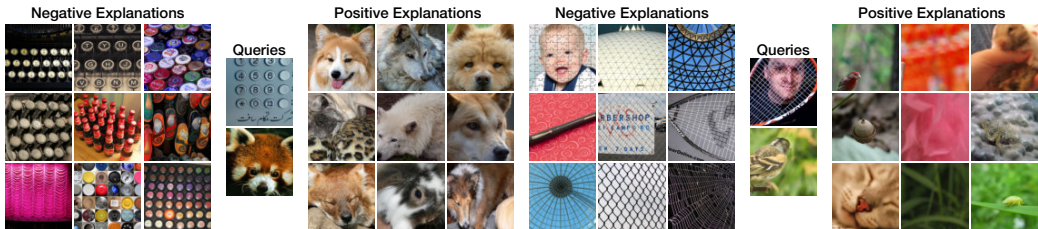

Fig. 9: **Examples of the 2-AFC Task.** For two different units of GoogLeNet one task each is shown. Every task contains a set of negative (left) and positive (right) visual explanations describing which visual feature the unit is sensitive to. In the center, two query images in the form of strongly and weakly activating dataset examples are shown, respectively. This means that each one of the two query images corresponds to the positive and the other to the negative explanations. The task is now to choose which query image corresponds to the positive ones.

## A.2 Task Construction

For constructing tasks, we follow Zimmermann et al. [50]. Specifically, this means that we use $K = 9$ (positive and negative) explanations in each task. We restrict explanations to natural dataset examples to reduce complexity but note that the same setup can also be applied to other visual explanations, such as feature visualizations. To choose query images and explanations, we proceed as follows: For each unit, we determine the $N \cdot (K + 1)$ most and least activating images, respectively. Out of these, the $N \cdot K$ most extreme images are used as explanations, the others as query images. The $N \cdot K$ potential explanation images are uniformly distributed across tasks according to their elicited activation level (see [6, 50] for more details).

# B   Psychophysical Evaluation

In Sec. 4.1.2, we validate the correctness of the proposed MIS in terms of estimating the interpretability for new, not previously analyzed, units. As described above, we use the MIS to create a set of the most and least interpretable units (42 units each). We then collect human annotations (i.e., human interpretability scores) for these units by running a psychophysical experiment. Specifically, we let humans solve the 2-AFC task described in Appx. A.1, originally introduced by [6]. We use the open-sourced setup of Zimmermann et al. [50] and recruit participants from Amazon Mechanical Turk. See Fig. 10 for screenshots of the task and the information given to participants on how their anonymized responses will be used. We leverage established quality/attention checks to control for high data quality and restrict the pool of potential participants to experienced workers [49, 50]. In total, we recruit 236 participants for our evaluation. The recruited workers are compensated at a targeted hourly rate of 15 USD, i.e., 2.79 USD per task based on the average task duration reported by Zimmermann et al. [50]. All procedures conform to Standard 8 of the American Psychological 405 Association's "Ethical Principles of Psychologists and Code of Conduct" (2016).

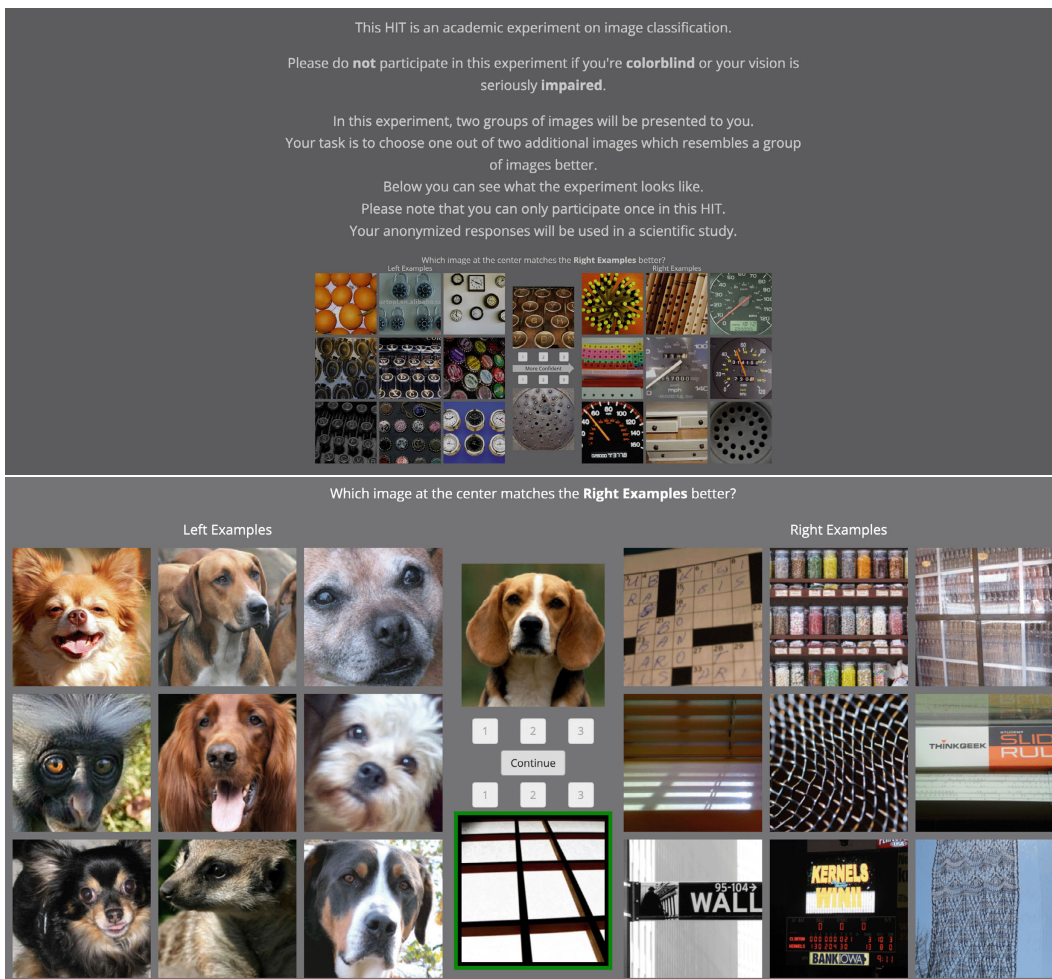

Fig. 10: Screenshot of the initial overview of the HIT presented to workers considering the task. We inform participants that they consent to their anonymized data being used for a scientific study.

## C  Influence of the Underlying Perceptual Similarity on the Machine Interpretability Score

As stated in Sec. 3, we used DreamSim [14] as the underlying perceptual similarity $f$ for all experiments shown so far. We now repeat the experiments on IMI in Sec. 4.1.1 with two alternative similarity measures: LPIPS [47] and DISTS [10]. While all three measures are based on learned image features, DreamSim leverages an ensemble of modern vision models trained on larger datasets compared to LPIPS and DISTS, which use AlexNet [24] and VGG16 [42] trained on ImageNet, respectively. According to Fu et al. [14], DreamSim clearly outperforms LPIPS and DISTS on image similarity benchmarks.

When comparing MIS based on DreamSim with one based on LPIPS and DISTS on a per-model level (see Fig. 11) one sees very similar results and strong correlations between each MIS and HIS. This might suggest that the choice of the similarity function to use has little influence on the quality of MIS. The picture, however, changes when zooming in and looking at per-unit interpretability (see Fig. 13). Now, it becomes evident that the MIS based on DreamSim outperforms that based on LPIPS and DISTS, indicated by the higher correlation and smaller spread of the point cloud. We, therefore, conclude that DreamSim is the best perceptual similarity available for computing machine interpretability scores.

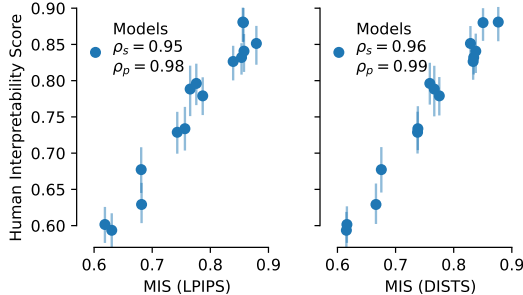

Fig. 11: **LPIPS and DISTS Perform Similarly as DreamSim when Comparing Models.** We compare DreamSim with two earlier perceptual similarity metrics, LPIPS and DISTS. All three lead to similar results on IMI (cf. Fig. 2A). See Fig. 13 for comparing these similarity functions on a per-unit level. standard deviation.

**Noise Ceiling of Annotations in IMI**   To put the difference in performance between the perceptual similarities on a per-unit level into context, we estimate the noise ceiling of the data: As the HIS for a single unit is a (potentially) noisy estimate over (up to 30) human decisions, it has some uncertainty. To account for this, we run a statistical simulation in which we model individual human responses as binary decisions from a Bernoulli distribution whose mean equals the unit's HIS. We can now simulate human decisions by sampling from the distribution. Then, we compute the correlation between MIS and simulated HIS and repeat the process 1 000 times. The resulting *noise ceiling* is compared to the correlations obtained when using LPIPS, DISTS, and DreamSim in Fig. 12. DreamSim's performance is very close to the noise ceiling for estimating the per-unit human interpretability.

## D  Sensitivity of the MIS on the Number of Tasks

As described in  Sec. 3, we compute the MIS by averaging over $N = 20$ tasks. This choice was initially motivated by previous work by Borowski et al. [6]. We investigate now how this choice influences the MIS. For this, we perform two experiments for GoogLeNet (see Fig. 14). First, we use the method for constructing tasks described before in Appx. A.2 to create 20 tasks per unit and then compute how the MIS changes when only using the first $i = 1, \ldots, 19$ tasks compared to all 20. While this setting is straightforward to analyze, it does not reflect how the number of tasks influences the MIS computation in practice: Using the task creation above, the chosen number of tasks influences the creation of all tasks, e.g., adding one more task changes which images are used for previous tasks. Therefore, in the second experiment, we again measure how the MIS changes

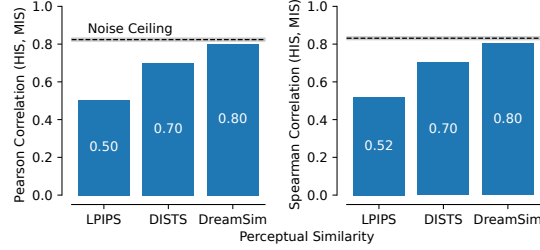

Fig. 12: **Best Perceptual Similarity Approaches Noise Ceiling.** Considering the noise ceiling, caused by the inherent uncertainty of the HIS, the best perceptual similarity (DreamSim) shows an almost perfect performance. The black bar and shaded area show the mean correlation and standard deviation over $1\,000$ simulations, respectively.

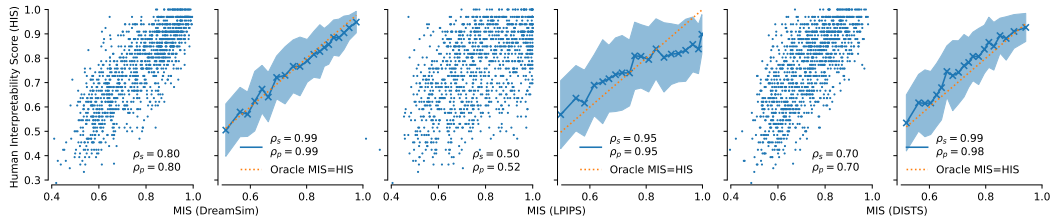

Fig. 13: **LPIPS and DISTS Perform Worse than DreamSim when Comparing Individual Units.** We compare DreamSim with two earlier perceptual similarity metrics, LPIPS and DISTS. While LPIPS and DISTS perform similarly to DreamSim on a per-model level of IMI (cf. Fig. 13), they lead to worse performance on a per-unit level.

when using $i = 1, \ldots, 19$ tasks compared to 20, but recreate all tasks when increasing their number. For both settings, we see that the residual converges to zero, with a slower convergence in the more realistic setting.

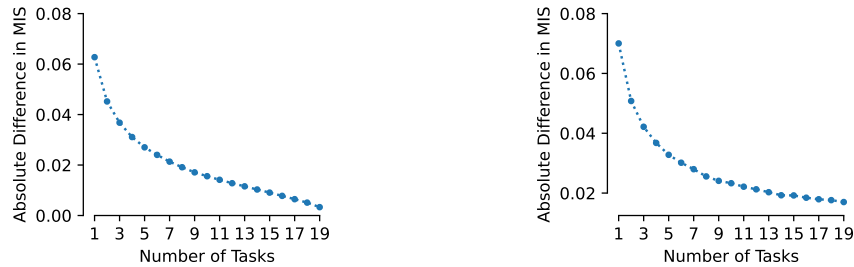

(a) New tasks do not influence earlier tasks.          (b) New tasks influence earlier tasks.

Fig. 14: **Convergence of MIS.** We investigate how MIS changes depending on the number of tasks $N$ that it is computed over. Here, we distinguish between two settings. In (a), we simulate that adding another task does not change the selection of query images and explanations in earlier tasks; in (b), this is not the case. While the former is easier to analyze due to a reduced level of randomness, note that the latter is the more relevant setting in practice. For both cases, we visualize the average absolute difference in MIS estimated for $< 20$ and $N = 20$ tasks.

# E  Applying MIS for Different Explanation Methods

The experiments in Sec. 4 compute the MIS for one type of explanation, namely strongly activating dataset examples. We now demonstrate that the same approach easily generalizes to other visual explanations: feature visualizations. We do not tune any hyperparameters but re-use the same as presented in Sec. 3 for dataset examples as explanations. In Fig. 15 we repeat the experiment from Fig. 2A and again see a strong correlation between MIS and HIS.

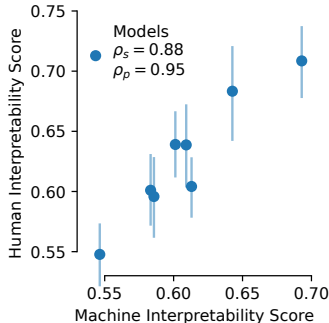

Fig. 15: **MIS Generalizes Well to Other Explanation Types.** We find a high correlation between MIS and HIS for other explanation types (feature visualizations). See Fig. 2A for the corresponding results for using natural dataset examples as explanations.

# F  Analysis of Constant Units

After training a network, it might happen that some of its units effectively become non-active/constant for any relevant image. We here call a unit *constant* if the difference between maximally and minimally elicited activation by the entire ImageNet-2012 training set is less than $10^{-8}$. As mentioned at the beginning of Sec. 4, we excluded those units in our analysis, as they do not present any interesting behavior that is worth understanding. Note that this does not mean that it will not be interesting to understand why such units exist. In Fig. 16, we display the ratio of constant units for each model. For most models, we see a low number of constant units: Specifically, we see that out of the 835 models investigated, 256 do not contain any constant units, 89 contain more than 1 % and 22 more than 5 %. Note that we here used the same notion of units as in the rest of the paper, meaning that we take the spatial mean of feature maps with spatial dimensions (e.g., for convolutional layers).

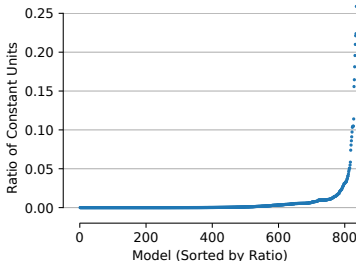

Fig. 16: **Ratio of Constant Units.** We compute the ratio of units constant with respect to the input (over the training set of ImageNet-2012) for all models considered. While the ratio is low for most models, it becomes large for a few models.

# G  Computational Resources

**Complexity of MIS**  Computing the MIS of a unit consists of four steps: (1) determining its visual explanations, (2) finding the strongly and weakly activating dataset samples to be used as the query images of the 2-AFC task, (3) computing the pairwise image similarities, and (4) computing the

final MIS. Due to the simplicity of the MIS' computation, its cost is neglectable. The complexity of the first step depends on the visualization method used: Gradient-based search algorithms, e.g., feature visualizations, require hundreds of forward and backward passes (of small batches), while determining dataset examples requires only a single forward pass over a sufficiently large dataset. The second step also mostly requires a single forward pass over this large dataset. Thus, if dataset examples are used as explanations, this step is free. Performing the third step requires computing the pairwise similarities of the images used in the created tasks. However, as most perceptual similarities, most importantly the leveraged DreamSim metric, are computed as the cosine similarity of an image's features, the step can be greatly simplified: We first compute and store the features for every image in the dataset used to sample the tasks' images. Then, computing the similarities equals only querying two features from a hash map and computing their cosine similarity. While this caching approach is not necessary for computing the MIS of a single unit, it becomes important when computing it for thousands of units. In this case, computing and caching the similarities also becomes neglectable, meaning that the computational cost of the MIS is dominated by the first and second steps. In summary, computing the MIS mostly resorts to a single forward pass over a sufficiently large dataset and additional forward/backward passes only depending on the visualization technique used.

**Resources Used**   Due to the aforementioned low computational complexity of the MIS, the experiments in Sec. 4 do not require much compute: Evaluating all units of a model takes, on average and varying depending on the model's size, less than one hour on a GPU (e.g., NVIDIA RTX 2080-TI or V100). Therefore, reproducing the experimental results of this paper requires approximately 1000 GPU hours.

## H   Impact Statement

This paper presents work whose goal is to advance the field of Machine Learning, specifically the field of Interpretable Machine Learning. The main contribution of our work is the presentation of a more time- and cost-efficient approach for quantifying how well humans can understand neural activations. A potential risk in automating interpretability research is that we will start optimizing for metrics that are never fully aligned with human judgments. It is conceivable that this will encourage the design of models that ace our metric but whose inner workings and decision-making processes are still obscure to human observers. This would set false goalposts and potentially come with safety risks if a high score in MIS were mistaken for a white box model that comes with higher trustworthiness. Beyond that, we see many potential use cases for this result (see Sec. 5), that can all advance the state of machine learning. There are potential societal consequences of our work, however, none of which we feel must be specifically highlighted here.

## I   Analyzing SAEs

Sparse Auto-Encoders (SAE) have been recently proposed as a means to understand the behavior of a network's layer better [7]: By finding a new, sparser basis to represent the layer's original activation, one hopes to find new artificial computational units that are more monosemantic. These units are expected to be easier to understand, rendering the tasks of understanding the behavior of the entire layer easier, too. While conceptually simple, the implementation and evaluation of SAEs is intricate: Training them requires careful hyperparameter tuning and algorithmic design choices such that the final SAEs are as sparse as possible but still faithful to the layer's original activations. However, as no reliable automatic interpretability evaluation has existed so far, evaluating SAEs in terms of how much more interpretable their features are is difficult, resulting in potentially inconclusive results. For example, Rajamanoharan et al. [39] suggested a modification to the usual SAE architecture (Gated SAE) but could not find a statistically significant benefit over the default architecture due to the high and, thus, prohibitive cost of interpretability evaluations.

As the MIS enables cheap interpretability evaluations, we can now pick up this work: In the context of vision models, we train different SAEs and Gated SAEs and compare their interpretability. Specifically, we train them on activations of one layer of GoogLeNet (mixed4b_3x3) and use different expansion factors and weights of the sparsity loss to obtain different SAEs. In addition to their interpretability, we also evaluate models in terms of their sparsity ($\ell_0$ count) and their reconstruction fidelity, i.e., how well they maintain the original model's classification cross-entropy compared to a

random model. In line with [39], Fig. 17 shows that Gated SAEs allow a better fidelity vs. sparsity trade-off. In terms of their MIS, we do not see a systematic difference between the two architectures. Moreover, in light of the high MIS of the original layer (i.e., 91.21 %), we do not see a strong benefit of SAEs compared to analyzing the original layer yet.

In another experiment, we trained (vanilla) SAEs on another, less interpretable layer (layer2_2_conv2 of a ResNet50). While the units of the original layer have an average MIS of 0.854 we observe that MIS values of up to 0.922 for SAEs with ten times more units than the original layer. See Tab. 1 for a sensitivity study on the relationship of an SAE's sparsity and its MIS.

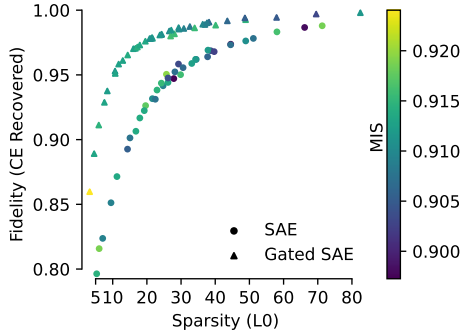

Fig. 17: **Comparable MIS for different SAE architectures**. We compare two types of SAEs used by Bricken et al. [7] and Rajamanoharan et al. [39] (SAE and Gated SAE, respectively), in terms of their sparsity, reconstruction fidelity and interpretability. While Gated SAEs allow a more optimal tradeoff between fidelity and sparsity, they are comparably interpretable as standard SAEs. The SAEs overall MIS is in a similar regime as the original layer's (91.21 %), while the sparsity is stronger than that of the original layer ($\approx 75$).

Tab. 17: **Sensitivity of SAE's MIS on its Hyperparameters.**

| Sparsity Weight $\lambda$ [$10^{-2}$] | 1.125 | 2.5 | 3.75 | 5.0 | 6.25 | 7.5 | 8.75 | 10.0 |
|---|---|---|---|---|---|---|---|---|
| L0 Count | 233 | 138 | 99 | 75 | 60 | 49 | 41 | 35 |
| MIS | 0.892 | 0.908 | 0.916 | 0.915 | 0.919 | 0.918 | 0.922 | 0.918 |

## J   Details on Models

In addition to the 9 models investigated by Zimmermann et al. [50] (GoogLeNet, ResNet-50, Clip ResNet-50, Robust (L2) ResNet-50, DenseNet-101, WideResNet-50, Clip ViT-B32, ViT-B32), we include one more model suggested by them (Robust (L2) ResNet-50) and 825 models from timm [44]:

xcit_tiny_12_p16_224.fb_in1k,     vit_tiny_patch16_384.augreg_in21k_ft_in1k,     pit_xs_224.in1k,     repghost-net_111.in1k,     regnetz_c16_evos.ch_in1k,     poolformer_m48.sail_in1k,     repghostnet_080.in1k,     volo_d3_448.sail_in1k, vit_base_patch16_224.augreg_in21k_ft_in1k,     regnety_320.tv2_in1k,     densenet121.ra_in1k,     mobilenetv3_large_100.ra_in1k, repghostnet_150.in1k,     seresnext26ts.ch_in1k,     regnety_160.swag_ft_in1k,     hrnet_w40.ms_in1k,     convnext_small.in12k_ft_in1k, vit_base_patch16_224.sam_in1k,     seresnextaa101d_32x8d.sw_in12k_ft_in1k_288,     vit_tiny_r_s16_p8_384.augreg_in21k_ft_in1k, regnety_320.pycls_in1k,     cs3darknet_m.c2ns_in1k,     vit_tiny_patch16_224.augreg_in21k_ft_in1k,     resnet101c.gluon_in1k,     con-vnextv2_atto.fcmae_ft_in1k,     flexivit_base.600ep_in1k,     xcit_small_12_p16_384.fb_dist_in1k,     mobilenetv2_050.lamb_in1k, flexivit_base.300ep_in1k,     resnext50_32x4d.tv_in1k,     resnet152.tv_in1k,     seresnext26d_32x4d.bt_in1k,     fbnetv3_g.ra2_in1k,     pool-former_s36.sail_in1k, resnext101_32x8d.tv_in1k, rexnet_130.nav_in1k, efficientvit_b2.r224_in1k, convnext_small.fb_in22k_ft_in1k_384, resnet50_gn.a1h_in1k,     eva02_small_patch14_336.mim_in22k_ft_in1k,     regnety_032.ra_in1k,     res2net50d.in1k,     convit_small.fb_in1k, regnetx_160.pycls_in1k,     convnextv2_large.fcmae_ft_in22k_in1k_384,     tf_efficientnet_b0.ns_jft_in1k,     pit_ti_224.in1k, volo_d1_384.sail_in1k, xcit_small_12_p8_384.fb_dist_in1k, dpn131.mx_in1k, resnext101_64x4d.gluon_in1k, densenet169.tv_in1k, resnet101d.ra2_in1k, repghostnet_200.in1k, resnet18.a2_in1k, xcit_small_12_p16_224.fb_in1k, pvt_v2_b3.in1k, dm_nfnet_f1.dm_in1k, vit_large_patch32_384.orig_in21k_ft_in1k,     convnextv2_tiny.fcmae_ft_in22k_in1k_384,     gcresnet50t.ra2_in1k,     nf_regnet_b1.ra2_in1k, volo_d1_224.sail_in1k, resnet50.ram_in1k, hrnet_w18_small_v2.ms_in1k, convnext_base.clip_laion2b_augreg_ft_in1k, regnetx_160.tv2_in1k, sequencer2d_l.in1k, convnext_large.fb_in22k_ft_in1k, botnet26t_256.c1_in1k, gc_efficientnetv2_rw_t.agc_in1k, wide_resnet50_2.racm_in1k,

halonet50ts.a1h_in1k, cspresnext50.ra_in1k, resnetv2_50d_evos.ah_in1k, tf_efficientnetv2_b3.in21k_ft_in1k, resnet152.gluon_in1k, lambda_resnet26rpt_256.c1_in1k, fastvit_sa24.apple_dist_in1k, xcit_medium_24_p8_384.fb_dist_in1k, repvit_m0_9.dist_450e_in1k, regnetx_320.pycls_in1k, seresnextaa101d_32x8d.sw_in12k_ft_in1k, efficientvit_b2.r288_in1k, convnext_tiny.in12k_ft_in1k, xcit_large_24_p16_384.fb_dist_in1k, resnetv2_50.a1h_in1k, coatnet_0_rw_224.sw_in1k, efficientnet_es_pruned.in1k, dla60_res2net.in1k, efficientformer_l7.snap_dist_in1k, cait_xxs24_224.fb_dist_in1k, vit_small_patch16_224.augreg_in21k_ft_in1k, tf_efficientnet_cc_b1_8e.in1k, efficientvit_b1.r288_in1k, halonet26t.a1h_in1k, mixnet_m.ft_in1k, hrnet_w44.ms_in1k, regnety_160.tv2_in1k, xcit_nano_12_p8_384.fb_dist_in1k, seresnext101_32x8d.ah_in1k, efficientvit_b2.r256_in1k, vit_base_patch16_clip_224.laion2b_ft_in12k_in1k, tf_efficientnet_lite2.in1k, deit3_small_patch16_224.fb_in1k, hrnet_w18_ssld.paddle_in1k, tf_efficientnet_b2.aa_in1k, crossvit_15_dagger_240.in1k, deit3_small_patch16_224.fb_in22k_ft_in1k, haloregnetz_b.ra3_in1k, tf_efficientnetv2_b0.in1k, eca_nfnet_l0.ra2_in1k, twins_pcpvt_small.in1k, ecaresnet50t.ra2_in1k, fastvit_sa12.apple_dist_in1k, skresnext50_32x4d.ra_in1k, resnet50d.a2_in1k, vit_base_patch32_clip_224.laion2b_ft_in1k, resnetblur50.bt_in1k, vit_base_patch16_224.orig_in21k_ft_in1k, resnet50.a1h_in1k, hardcorenas_e.miil_green_in1k, coatnext_nano_rw_224.sw_in1k, convnext_base.clip_laiona_augreg_ft_in1k_384, tresnet_m.miil_in1k_448, resnet10t.c3_in1k, poolformerv2_m48.sail_in1k, tf_efficientnet_b1.aa_in1k, edgenext_base.usi_in1k, tf_efficientnet_es.in1k, tresnet_l.miil_in1k_448, resnet152.a1h_in1k, mixnet_s.ft_in1k, resnet50.am_in1k, rexnet_100.nav_in1k, xcit_large_24_p8_224.fb_dist_in1k, deit3_base_patch16_224.fb_in22k_ft_in1k, xcit_tiny_24_p8_384.fb_dist_in1k, coat_lite_medium_384.in1k, focalnet_small_srf.ms_in1k, vit_base_patch8_224.augreg_in21k_ft_in1k, convnext_tiny_hnf.a2h_in1k, visformer_small.in1k, vit_small_r26_s32_384.augreg_in21k_ft_in1k, vgg16_bn.tv_in1k, eca_nfnet_l1.ra2_in1k, xcit_small_12_p8_224.fb_in1k, beitv2_base_patch16_224.in1k_ft_in22k_in1k, cs3edgenet_x.c2_in1k, vit_base_patch16_clip_384.laion2b_ft_in12k_in1k, xcit_small_12_p16_224.fb_dist_in1k, convformer_b36.sail_in1k_384, bat_resnext26ts.ch_in1k, caformer_b36.sail_in1k, dla34.in1k, crossvit_18_dagger_240.in1k, tf_efficientnetv2_s.in21k_ft_in1k, focalnet_base_srf.ms_in1k, convformer_b36.sail_in22k_ft_in1k_384, resnet34.tv_in1k, resmlp_24_224.fb_distilled_in1k, convnext_base.clip_laion2b_augreg_ft_in12k_in1k, caformer_s18.sail_in1k_384, resnetaa50.a1h_in1k, beitv2_base_patch16_224.in1k_ft_in1k, convformer_m36.sail_in22k_ft_in1k, inception_resnet_v2.tf_ens_adv_in1k, mobilenetv2_110d.ra_in1k, resnext101_32x4d.fb_swsl_ig1b_ft_in1k, regnetx_008.tv2_in1k, convnext_small.in12k_ft_in1k_384, levit_conv_128.fb_dist_in1k, volo_d3_224.sail_in1k, nest_tiny_jx.goog_in1k, mobileone_s2.apple_in1k, fastvit_t8.apple_dist_in1k, halo2botnet50ts_256.a1h_in1k, mobilenetv2_140.ra_in1k, caformer_m36.sail_in1k, seresnet50.ra2_in1k, hardcorenas_d.miil_green_in1k, convformer_b36.sail_in1k, regnety_320.swag_ft_in1k, volo_d4_448.sail_in1k, tf_efficientnet_b2.ns_jft_in1k, sebotnet33ts_256.a1h_in1k, vit_small_patch32_224.augreg_in21k_ft_in1k, vit_base_patch32_224.sam_in1k, resnetv2_50d_gn.ah_in1k, mobileone_s4.apple_in1k, coat_small.in1k, tf_mixnet_l.in1k, resnet34.a2_in1k, regnetx_032.pycls_in1k, resnetaa101d.sw_in12k_ft_in1k, lcnet_100.ra2_in1k, repvgg_b1.rvgg_in1k, crossvit_15_240.in1k, edgenext_x_small.in1k, repvit_m1_5.dist_300e_in1k, hardcorenas_a.miil_green_in1k, efficientformer_l1.snap_dist_in1k, tf_mobilenetv3_large_075.in1k, hrnet_w18_small.ms_in1k, tf_efficientnet_b2.in1k, ghostnetv2_130.in1k, ecaresnet26t.ra2_in1k, fastvit_s12.apple_in1k, xcit_tiny_12_p8_224.fb_dist_in1k, tresnet_m.miil_in21k_ft_in1k, fastvit_sa24.apple_in1k, resnetrs200.tf_in1k, convnextv2_nano.fcmae_ft_in1k, resnet50.ra_in1k, resnet34.bt_in1k, regnety_002.pycls_in1k, focalnet_base_lrf.ms_in1k, dla102.in1k, regnetz_e8.ra3_in1k, pvt_v2_b0.in1k, xcit_medium_24_p8_224.fb_in1k, regnety_640.seer_ft_in1k, resnet200d.ra2_in1k, caformer_s36.sail_in1k_384, deit3_small_patch16_384.fb_in22k_ft_in1k, eca_resnet26ts.ch_in1k, vgg13.tv_in1k, tf_efficientnet_lite0.in1k, resnet50.b1k_in1k, dla60_res2next.in1k, repvit_m1_1.dist_300e_in1k, convnext_base.fb_in22k_ft_in1k, tf_efficientnet_cc_b0_4e.in1k, ese_vovnet19b_dw.ra_in1k, resnetv2_152x2_bit.goog_teacher_in21k_ft_in1k, deit_base_distilled_patch16_384.fb_in1k, resnet101d.gluon_in1k, convnext_large.fb_in22k_ft_in1k_384, darknet53.c2ns_in1k, poolformerv2_s36.sail_in1k, convformer_m36.sail_in22k_ft_in1k_384, gmlp_s16_224.ra3_in1k, convformer_s18.sail_in1k, efficientnet_em.ra2_in1k, inception_v3.gluon_in1k, resmlp_12_224.fb_in1k, tresnet_l.miil_in1k, ecaresnet101d_pruned.miil_in1k, resnet152.a2_in1k, vit_small_patch32_384.augreg_in21k_ft_in1k, inception_v3.tf_adv_in1k, repghostnet_130.in1k, levit_conv_384.fb_dist_in1k, repvit_m1_5.dist_450e_in1k, efficientnet_el.ra_in1k, seresnet50.a2_in1k, pit_s_distilled_224.in1k, cspdarknet53.ra_in1k, tf_efficientnet_cc_b0_8e.in1k, densenet201.tv_in1k, resnext50_32x4d.a1_in1k, cs3sedarknet_l.c2ns_in1k, cait_s24_384.fb_dist_in1k, spnasnet_100.rmsp_in1k, res2net50_14w_8s.in1k, repvgg_d2se.rvgg_in1k, regnetx_032.tv2_in1k, crossvit_18_dagger_408.in1k, pit_b_distilled_224.in1k, cs3darknet_focus_l.c2ns_in1k, resnet50.bt_in1k, vgg11.tv_in1k, convnextv2_femto.fcmae_ft_in1k, convnext_nano.in12k_ft_in1k, resnext101_64x4d.tv_in1k, convnext_nano.d1h_in1k, cspresnet50.ra_in1k, tf_mixnet_m.in1k, xcit_tiny_12_p16_384.fb_dist_in1k, seresnet50.a1_in1k, efficientnetv2_rw_t.ra2_in1k, resnet152d.gluon_in1k, regnety_032.tv2_in1k, inception_resnet_v2.tf_in1k, eva_large_patch14_196.in22k_ft_in1k, pvt_v2_b1.in1k, convformer_m36.sail_in1k_384, densenet161.tv_in1k, dla102x.in1k, edgenext_small_rw.sw_in1k, regnety_016.tv2_in1k, convnextv2_base.fcmae_ft_in1k, vit_large_patch14_clip_336.laion2b_ft_in12k_in1k, levit_conv_128s.fb_dist_in1k, hrnet_w48.ms_in1k, resnet101.a1h_in1k, xcit_medium_24_p8_224.fb_dist_in1k, resnetrs152.tf_in1k, convnextv2_nano.fcmae_ft_in22k_in1k, convnextv2_tiny.fcmae_ft_in22k_in1k, resnext50_32x4d.bt_in1k, gernet_s.idstcv_in1k, selecsls42b.in1k, repvit_m3.dist_in1k, resnest50d_1s4x24d.in1k, dpn98.mx_in1k, xcit_nano_12_p16_224.fb_in1k, regnetx_016.pycls_in1k, xcit_medium_24_p16_224.fb_in1k, caformer_s18.sail_in1k, sehalonet33ts.ra2_in1k, tinynet_c.in1k, xcit_tiny_24_p16_224.fb_dist_in1k, flexivit_small.300ep_in1k, resnext101_32x8d.tv2_in1k, convnextv2_base.fcmae_ft_in22k_in1k_384, semnasnet_075.rmsp_in1k, res2net50_26w_4s.in1k, cait_xxs24_384.fb_dist_in1k, mobilenetv2_120d.ra_in1k, seresnext26t_32x4d.bt_in1k, flexivit_base.1200ep_in1k, res2net50_26w_6s.in1k, vit_base_patch16_clip_384.openai_ft_in12k_in1k, nest_base_jx.goog_in1k, ecaresnetlight.miil_in1k, repvgg_b0.rvgg_in1k, ecaresnet50t.a1_in1k, inception_next_tiny.sail_in1k, regnety_032.pycls_in1k, mixer_b16_224.miil_in21k_ft_in1k, poolformer_s12.sail_in1k, vit_base_patch32_clip_384.openai_ft_in12k_in1k, vit_base_patch32_384.augreg_in21k_ft_in1k, efficientvit_b1.r224_in1k, vit_base_patch16_clip_384.laion2b_ft_in1k, deit_small_distilled_patch16_224.fb_in1k, efficientvit_b0.r224_in1k, resnest50d.in1k, regnety_120.pycls_in1k, semnasnet_100.rmsp_in1k, wide_resnet50_2.tv_in1k, xcit_small_24_p16_224.fb_in1k,

resnet101.a3_in1k, fastvit_t12.apple_in1k, tf_efficientnet_lite1.in1k, tinynet_a.in1k, resmlp_big_24_224.fb_distilled_in1k, cs3se_edgenet_x.c2ns_in1k, resnetv2_152x2_bit.goog_teacher_in21k_ft_in1k_384, resnext50_32x4d.tv2_in1k, efficientnet_b2.ra_in1k, convformer_s18.sail_in22k_ft_in1k_384, caformer_s18.sail_in22k_ft_in1k_384, deit3_base_patch16_224.fb_in1k, vit_base_patch32_clip_384.laion2b_ft_in12k_in1k, vit_medium_patch16_gap_384.sw_in12k_ft_in1k, sequencer2d_s.in1k, mobileone_s0.apple_in1k, edgenext_base.in21k_ft_in1k, deit3_medium_patch16_224.fb_in1k, efficientformerv2_l.snap_dist_in1k, lambda_resnet50ts.a1h_in1k, xception41p.ra3_in1k, resnext50_32x4d.a3_in1k, crossvit_small_240.in1k, repvgg_a1.rvgg_in1k, resnet51q.ra2_in1k, xcit_small_24_p16_384.fb_dist_in1k, vit_base_patch32_clip_224.openai_ft_in1k, flexivit_large.300ep_in1k, repvgg_b3g4.rvgg_in1k, resnext50_32x4d.a1h_in1k, coat_lite_medium.in1k, vit_base_patch32_clip_448.laion2b_ft_in12k_in1k, resnext50_32x4d.gluon_in1k, repvgg_b2.rvgg_in1k, vit_base_patch16_rpn_224.sw_in1k, mixer_b16_224.goog_in21k_ft_in1k, resnet50.c2_in1k, lamhalobotnet50ts_256.a1h_in1k, tiny_vit_21m_512.dist_in22k_ft_in1k, xcit_large_24_p16_224.fb_dist_in1k, repvgg_a2.rvgg_in1k, gernet_l.idstcv_in1k, mobilevitv2_050.cvnets_in1k, convnextv2_base.fcmae_ft_in22k_in1k, resnet18.a3_in1k, ecaresnet50d.miil_in1k, coat_lite_small.in1k, convnext_xlarge.fb_in22k_ft_in1k, mobilevitv2_075.cvnets_in1k, cait_s36_384.fb_dist_in1k, efficientformerv2_s1.snap_dist_in1k, resnet18.fb_swsl_ig1b_ft_in1k, mobileone_s1.apple_in1k, resnet61q.ra2_in1k, tf_efficientnetv2_b3.in1k, mobilevitv2_175.cvnets_in1k, convnext_tiny.fb_in22k_ft_in1k_384, crossvit_tiny_240.in1k, caformer_b36.sail_in22k_ft_in1k_384, resnet152d.ra2_in1k, convit_base.fb_in1k, tinynet_b.in1k, deit3_large_patch16_384.fb_in22k_ft_in1k, regnetx_004_tv.tv2_in1k, cait_xxs36_384.fb_dist_in1k, convnext_nano_ols.d1h_in1k, efficientnet_lite0.ra_in1k, inception_v4.tf_in1k, hrnet_w18.ms_in1k, gernet_m.idstcv_in1k, convformer_s36.sail_in22k_ft_in1k_384, deit_tiny_distilled_patch16_224.fb_in1k, deit_small_patch16_224.fb_in1k, vit_large_patch14_clip_336.laion2b_ft_in1k, crossvit_18_240.in1k, resnet26.bt_in1k, resnet18.a1_in1k, deit3_base_patch16_384.fb_in22k_ft_in1k, convformer_s36.sail_in1k, convnext_small.fb_in22k_ft_in1k, selecsls60b.in1k, efficientnet_b0.ra_in1k, focalnet_tiny_srf.ms_in1k, ecaresnet101d.miil_in1k, regnetx_080.tv2_in1k, mobileone_s3.apple_in1k, mobilenetv3_rw.rmsp_in1k, poolformerv2_m36.sail_in1k, seresnextaa101d_32x8d.ah_in1k, levit_conv_192.fb_dist_in1k, focalnet_tiny_lrf.ms_in1k, regnety_320.swag_lc_in1k, tresnet_v2_l.miil_in21k_ft_in1k, seresnet50.a3_in1k, dla46x_c.in1k, cs3darknet_x.c2ns_in1k, tf_efficientnet_b0.ap_in1k, vit_base_patch16_224.augreg2_in21k_ft_in1k, resnext101_32x8d.fb_ssl_yfcc100m_ft_in1k, xcit_large_24_p8_384.fb_dist_in1k, tinynet_e.in1k, cait_xs24_384.fb_dist_in1k, fastvit_sa12.apple_in1k, hrnet_w64.ms_in1k, regnety_016.pycls_in1k, wide_resnet101_2.tv2_in1k, beitv2_large_patch16_224.in1k_ft_in22k_in1k, hrnet_w30.ms_in1k, resnet101.tv_in1k, repvit_m2.dist_in1k, coatnet_nano_rw_224.sw_in1k, flexivit_small.1200ep_in1k, tf_efficientnet_b0.in1k, tf_efficientnet_b1.in1k, efficientformer_l3.snap_dist_in1k, vit_base_patch16_384.augreg_in21k_ft_in1k, xcit_tiny_24_p8_224.fb_dist_in1k, dla102x2.in1k, hardcorenas_f.miil_green_in1k, regnety_064.ra3_in1k, resnext101_32x4d.gluon_in1k, tf_efficientnetv2_b2.in1k, resnet32ts.ra2_in1k, xcit_tiny_12_p8_384.fb_dist_in1k, inception_v3.tv_in1k, xcit_large_24_p16_224.fb_dist_in1k, ecaresnet50t.a3_in1k, repvit_m2_3.dist_450e_in1k, fbnetv3_b.ra2_in1k, vit_base_patch8_224.augreg2_in21k_ft_in1k, cs3darknet_l.c2ns_in1k, convnext_base.clip_laion2b_augreg_ft_in12k_in1k_384, regnety_160.deit_in1k, regnety_160.pycls_in1k, dla60x.in1k, xcit_tiny_24_p16_384.fb_dist_in1k, eva02_tiny_patch14_336.mim_in22k_ft_in1k, volo_d2_224.sail_in1k, regnety_160.swag_lc_in1k, vit_base_patch32_clip_224.laion2b_ft_in12k_in1k, tf_mixnet_s.in1k, repvit_m1_0.dist_300e_in1k, convnextv2_large.fcmae_ft_in1k, resmlp_12_224.fb_distilled_in1k, xcit_medium_24_p16_384.fb_dist_in1k, regnety_080_tv.tv2_in1k, dpn107.mx_in1k, inception_v3.tf_in1k, dpn68.mx_in1k, efficientnet_es.ra_in1k, mnasnet_100.rmsp_in1k, resnet101.tv2_in1k, res2next50.in1k, vit_base_patch16_clip_384.openai_ft_in1k, tf_efficientnet_b1.ns_jft_in1k, flexivit_small.600ep_in1k, visformer_tiny.in1k, resnet50.a1_in1k, dla60.in1k, regnetz_d32.ra3_in1k, senet154.gluon_in1k, efficientnetv2_rw_s.ra2_in1k, focalnet_small_lrf.ms_in1k, seresnet33ts.ra2_in1k, fbnetc_100.rmsp_in1k, resnet18d.ra2_in1k, resnet34.a3_in1k, dla60x_c.in1k, efficientnet_b1_pruned.in1k, efficientformerv2_s2.snap_dist_in1k, resnet50s.gluon_in1k, resnet101.a2_in1k, regnety_040.ra3_in1k, convmixer_1536_20.in1k, regnety_008_tv.tv2_in1k, resnet152.a1_in1k, mixnet_l.ft_in1k, gcresnext26ts.ch_in1k, vit_base_patch16_clip_224.openai_ft_in1k, fastvit_ma36.apple_in1k, vgg16.tv_in1k, gcresnext50ts.ch_in1k, xcit_tiny_12_p16_224.fb_dist_in1k, regnety_008.pycls_in1k, resmlp_36_224.fb_distilled_in1k, regnetz_040_h.ra3_in1k, inception_next_base.sail_in1k, dm_nfnet_f0.dm_in1k, resnet50.d_in1k, efficientnet_b2_pruned.in1k, resnet18.tv_in1k, rexnet_150.nav_in1k, convnext_large_mlp.clip_laion2b_soup_ft_in12k_in1k_320, ghostnetv2_160.in1k, vit_small_patch16_384.augreg_in21k_ft_in1k, convnext_xlarge.fb_in22k_ft_in1k_384, mobilenetv3_small_075.lamb_in1k, regnetz_d8_evos.ch_in1k, dm_nfnet_f3.dm_in1k, repvgg_b3.rvgg_in1k, convnext_large_mlp.clip_laion2b_augreg_ft_in1k_384, dpn68b.mx_in1k, resnext101_32x8d.fb_wsl_ig1b_ft_in1k, deit3_large_patch16_384.fb_in1k, convformer_s18.sail_in1k_384, repghostnet_058.in1k, fastvit_sa36.apple_dist_in1k, resnext50_32x4d.a2_in1k, regnetx_040.pycls_in1k, vit_base_r50_s16_384.orig_in21k_ft_in1k, vit_base_patch16_clip_224.laion2b_ft_in1k, deit3_base_patch16_384.fb_in1k, tf_efficientnetv2_s.in1k, ecaresnet50t.a2_in1k, resnetrs50.tf_in1k, gmixer_24_224.ra3_in1k, resnetaa50d.sw_in12k_ft_in1k, tresnet_xl.miil_in1k, resnest101e.in1k, regnetx_004.pycls_in1k, mnasnet_small.lamb_in1k, repvgg_a0.rvgg_in1k, resnetv2_50x1_bit.goog_in21k_ft_in1k, cait_s24_224.fb_dist_in1k, regnety_004.tv2_in1k, convnext_base.fb_in22k_ft_in1k_384, convnext_tiny.fb_in22k_ft_in1k_384, eca_halonext26ts.c1_in1k, resnet18.gluon_in1k, fastvit_s12.apple_dist_in1k, deit_base_patch16_224.fb_in1k, hrnet_w18.ms_aug_in1k, resnet33ts.ra2_in1k, seresnext101_64x4d.gluon_in1k, convnext_small.fb_in1k, convformer_s36.sail_in1k_384, pit_ti_distilled_224.in1k, resnet50.tv2_in1k, nest_small_jx.goog_in1k, resmlp_36_224.fb_in1k, hrnet_w18_small.gluon_in1k, vit_base_patch16_384.augreg_in1k, resnet50.fb_swsl_ig1b_ft_in1k, poolformer_m36.sail_in1k, tf_mobilenetv3_small_100.in1k, regnety_040.pycls_in1k, gcresnet33ts.ra2_in1k, resnet101s.gluon_in1k, darknetaa53.c2ns_in1k, poolformerv2_s12.sail_in1k, resnext50_32x4d.fb_ssl_yfcc100m_ft_in1k, poolformerv2_s24.sail_in1k, eca_resnet33ts.ra2_in1k, repvit_m2_3.dist_300e_in1k, nf_resnet50.ra2_in1k, convnext_pico_ols.d1_in1k, caformer_s36.sail_in1k, regnetz_040.ra3_in1k, vit_small_r26_s32_224.augreg_in21k_ft_in1k, resnext26ts.ra2_in1k, mixnet_xl.ra_in1k, deit_base_patch16_384.fb_in1k, repvit_m1_0.dist_450e_in1k, convmixer_1024_20_ks9_p14.in1k, regnety_064.pycls_in1k,

resnet34.gluon_in1k, res2net101_26w_4s.in1k, nfnet_l0.ra2_in1k, resnet34d.ra2_in1k, convnextv2_nano.fcmae_ft_in22k_in1k_384, twins_pcpvt_base.in1k, resnetv2_101.a1h_in1k, xcit_nano_12_p8_224.fb_dist_in1k, xcit_small_24_p8_224.fb_dist_in1k, resnet50.b2k_in1k, deit3_small_patch16_384.fb_in1k, hardcorenas_c.miil_green_in1k, coat_lite_mini.in1k, resnet152.tv2_in1k, densenetblur121d.ra_in1k, hrnet_w18_small_v2.gluon_in1k, vit_base_patch16_384.orig_in21k_ft_in1k, xcit_small_12_p8_224.fb_dist_in1k, convformer_m36.sail_in1k, xcit_nano_12_p16_384.fb_dist_in1k, resnet34.a1_in1k, convnext_atto_ols.a2_in1k, resnet14t.c3_in1k, twins_pcpvt_large.in1k, resnest26d.gluon_in1k, mobilenetv3_small_100.lamb_in1k, efficientnet_b3_pruned.in1k, vit_small_patch16_224.augreg_in1k, convnext_tiny.fb_in1k, resnet50d.a3_in1k, mobilevitv2_175.cvnets_in22k_ft_in1k_384, deit3_medium_patch16_224.fb_in22k_ft_in1k, seresnext101_32x4d.gluon_in1k, hardcorenas_b.miil_green_in1k, caformer_m36.sail_in22k_ft_in1k, ghostnetv2_100.in1k, ecaresnet50d_pruned.miil_in1k, caformer_s36.sail_in22k_ft_in1k_384, deit_tiny_patch16_224.fb_in1k, fastvit_sa36.apple_in1k, regnety_320.seer_ft_in1k, edgenext_small.usi_in1k, resmlp_big_24_224.fb_in22k_ft_in1k, regnety_160.lion_in12k_ft_in1k, regnety_160.sw_in12k_ft_in1k, tf_efficientnet_b1.ap_in1k, res2net50_48w_2s.in1k, eca_botnext26ts_256.c1_in1k, xcit_small_24_p8_224.fb_in1k, crossvit_9_dagger_240.in1k, coat_lite_tiny.in1k, resnetv2_101x1_bit.goog_in21k_ft_in1k, convnext_large_mlp.clip_laion2b_augreg_ft_in1k, xcit_nano_12_p16_224.fb_dist_in1k, cs3darknet_focus_m.c2ns_in1k, wide_resnet50_2.tv2_in1k, vit_base_patch16_clip_224.openai_ft_in12k_in1k, skresnet34.ra_in1k, repvgg_b1g4.rvgg_in1k, vgg19_bn.tv_in1k, repghostnet_100.in1k, regnetv_064.ra3_in1k, mobilenetv2_100.ra_in1k, convnext_femto.d1_in1k, resnet26t.ra2_in1k, regnetv_040.ra3_in1k, skresnet18.ra_in1k, caformer_m36.sail_in22k_ft_in1k_384, vit_base_patch32_384.augreg_in1k, regnetz_b16.ra3_in1k, hrnet_w48_ssld.paddle_in1k, resnest50d_4s2x40d.in1k, cait_xxs36_224.fb_dist_in1k, regnetx_016.tv2_in1k, xcit_small_24_p8_384.fb_dist_in1k, vit_tiny_r_s16_p8_224.augreg_in21k_ft_in1k, coat_mini.in1k, xcit_small_24_p16_224.fb_dist_in1k, caformer_s36.sail_in22k_ft_in1k, poolformer_s24.sail_in1k, resmlp_big_24_224.fb_in1k, regnetx_120.pycls_in1k, regnetz_d8.ra3_in1k, resnet50d.ra2_in1k, repvit_m1.dist_in1k, eca_nfnet_l2.ra3_in1k, resnet50d.gluon_in1k, seresnext50_32x4d.racm_in1k, vit_small_patch16_384.augreg_in1k, coat_tiny.in1k, xcit_nano_12_p8_224.fb_in1k, crossvit_base_240.in1k, resnet50d.a1_in1k, convformer_s36.sail_in22k_ft_in1k, convnextv2_large.fcmae_ft_in22k_in1k, resnet50.tv_in1k, resnet50.c1_in1k, pit_xs_distilled_224.in1k, efficientnet_b1.ft_in1k, tf_efficientnet_el.in1k, hrnet_w32.ms_in1k, vit_base_patch16_224_miil.in21k_ft_in1k, cs3sedarknet_x.c2ns_in1k, dpn68b.ra_in1k, tf_efficientnetv2_b1.in1k, regnety_004.pycls_in1k, tf_mobilenetv3_large_minimal_100.in1k, resnetrs101.tf_in1k, ese_vovnet39b.ra_in1k, mixer_l16_224.goog_in21k_ft_in1k, repghostnet_050.in1k, repvgg_b2g4.rvgg_in1k, repvit_m1_1.dist_450e_in1k, vit_base_patch32_224.augreg_in21k_ft_in1k, tf_mobilenetv3_large_100.in1k, pit_s_224.in1k, caformer_s18.sail_in22k_ft_in1k, wide_resnet101_2.tv_in1k, fastvit_t12.apple_dist_in1k, convmixer_768_32.in1k, vit_base_patch32_224.augreg_in1k, efficientformerv2_s0.snap_dist_in1k, resnest200e.in1k, levit_conv_256.fb_dist_in1k, resnet18.fb_ssl_yfcc100m_ft_in1k, vgg13_bn.tv_in1k, resnet152c.gluon_in1k, dla169.in1k, pvt_v2_b4.in1k, crossvit_15_dagger_408.in1k, convnext_femto_ols.d1_in1k, convnext_large.fb_in1k, regnetx_064.pycls_in1k, fastvit_t8.apple_in1k, seresnet152d.ra2_in1k, vgg19.tv_in1k, vgg11_bn.tv_in1k, dm_nfnet_f2.dm_in1k, seresnext101d_32x8d.ah_in1k, inception_next_base.sail_in1k_384, lambda_resnet26t.c1_in1k, resnetv2_152x2_bit.goog_in21k_ft_in1k, fastvit_ma36.apple_dist_in1k, regnety_006.pycls_in1k, regnety_080.pycls_in1k, resnet50.fb_ssl_yfcc100m_ft_in1k, tf_mobilenetv3_small_075.in1k, regnetz_c16.ra3_in1k, edgenext_xx_small.in1k, crossvit_9_240.in1k, xcit_tiny_24_p8_224.fb_in1k, regnety_080.ra3_in1k, efficientvit_b1.r256_in1k, tinynet_d.in1k, caformer_b36.sail_in1k_384, pvt_v2_b2.in1k, resnet26d.bt_in1k, convnext_pico.d1_in1k, pit_b_224.in1k, convnextv2_pico.fcmae_ft_in1k, fbnetv3_d.ra2_in1k, flexivit_large.1200ep_in1k, resnet50c.gluon_in1k, regnetx_080.pycls_in1k, convnext_base.fb_in1k, tf_efficientnet_em.in1k, vit_base_patch16_224.augreg_in1k, convit_tiny.fb_in1k, resnext50_32x4d.fb_swsl_ig1b_ft_in1k, dm_nfnet_f4.dm_in1k, resnet50.a3_in1k, convnext_atto.d2_in1k, efficientnet_el_pruned.in1k, volo_d2_384.sail_in1k, resnext101_32x4d.fb_ssl_yfcc100m_ft_in1k, repvit_m0_9.dist_300e_in1k, regnety_120.sw_in12k_ft_in1k, beit_base_patch16_384.in22k_ft_in22k_in1k, mobilenetv3_large_100.miil_in21k_ft_in1k, tf_efficientnet_b0.aa_in1k, inception_next_small.sail_in1k, deit_base_distilled_patch16_224.fb_in1k, lcnet_075.ra2_in1k, xcit_tiny_12_p8_224.fb_in1k, resnet101.gluon_in1k, dpn92.mx_in1k, resnet101.a1_in1k, selecsls60.in1k, beit_base_patch16_224.in22k_ft_in22k_in1k, convnextv2_tiny.fcmae_ft_in1k, res2net50_26w_8s.in1k, sequencer2d_m.in1k, vit_medium_patch16_gap_256.sw_in12k_ft_in1k, regnetx_008.pycls_in1k, resnet50.a2_in1k, res2net101d.in1k, vit_large_patch16_384.augreg_in21k_ft_in1k, pvt_v2_b2_li.in1k, regnetx_006.pycls_in1k, xcit_tiny_24_p16_224.fb_in1k, pvt_v2_b5.in1k, resnext50_32x4d.ra_in1k, resnest14d.gluon_in1k, caformer_m36.sail_in1k_384, resnet50.gluon_in1k, resnet152s.gluon_in1k, flexivit_large.600ep_in1k, resnetv2_50x1_bit.goog_distilled_in1k, resmlp_24_224.fb_in1k, deit3_large_patch16_224.fb_in1k, seresnext50_32x4d.gluon_in1k, densenet121.tv_in1k, resnet152.a3_in1k, ghostnet_100.in1k, tf_efficientnet_b2.ap_in1k, regnetx_002.pycls_in1k.

# K  Additional Results

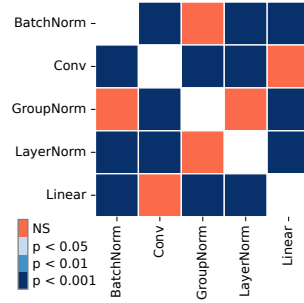

Fig. 18: **Differences Between Layer Types are Significant.** We analyze and test for statistical significances in the differences in MIS between different layer types (see Fig. 5. The reported significance levels were computed using Conover's test over the per-model and per-layer-type means with Holm's correction for multiple comparisons.

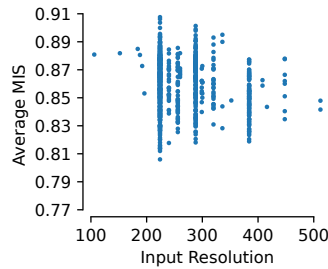

Fig. 19: **Influence of Input Resolution of MIS.** We show the average MIS per model as a function of the model's input resolution. No trend is apparent; models with the same resolution yield different interpretability levels.

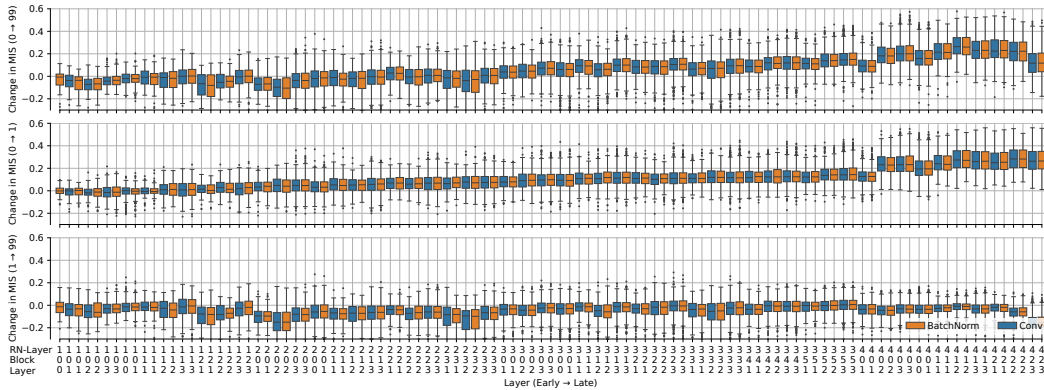

Fig. 20: **Change of Interpretability per Layer During Training.** Detailed version of Fig. 8.

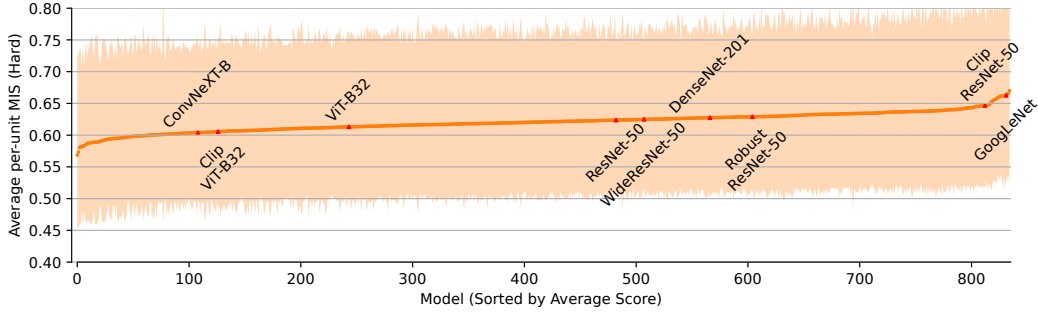

Fig. 21: **Comparison of the Average Per-unit MIS for Models for a Different Task Difficulty.** Our proposed MIS can easily be extended to test more than just the extrema of the activation distribution: Instead of choosing the most extremely activating samples as query images, we can sample less strongly activating ones from other parts of the activation distribution. By sampling from the 2nd/98th percentile, we can recompute Fig. 3 on a more challenging version of the underlying 2-AFC task.

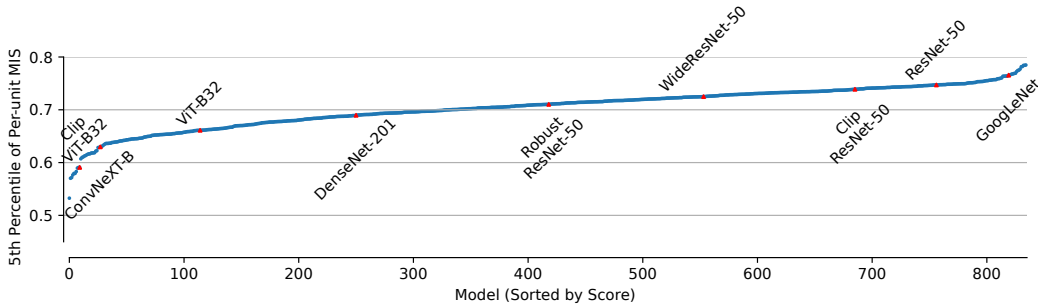

Fig. 22: **Comparison of the Minimum of the Per-unit MIS for Models.** While the mean of the per-unit interpretability varies in a rather narrow value range (see Fig. 3), we investigate differences in the distribution of scores. Specifically, we are interested in the effective width of the distribution, i.e., how low does the minimal MIS per model go? To make the analysis robust against outliers, we do not use the minimum but instead the 5th percentile. Note that this corresponds to the lower end of the shaded area in Fig. 3. Compared to the average MIS, we see higher variability across models.

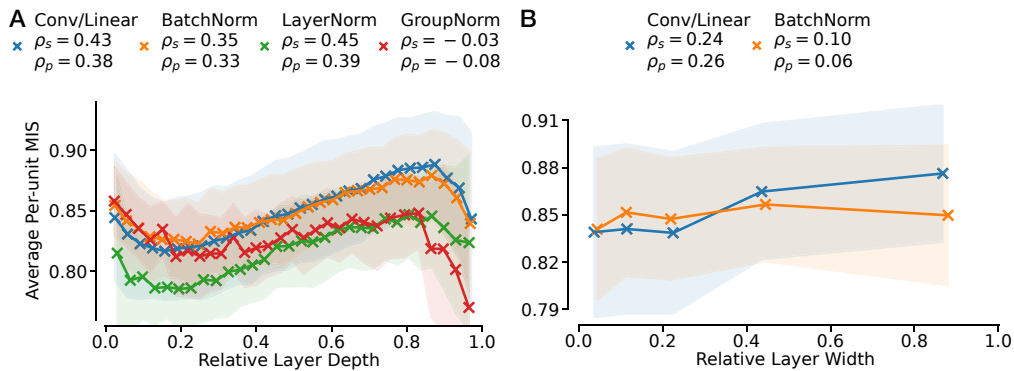

Fig. 23: **(A) Deeper Layers are More Interpretable.** Average MIS per layer as a function of the relative depth of the layer within the network, grouped by layer types. For each type, the values are grouped into 30 bins of equal count based on the relative depth. The markers shown correspond to the bin average, the shaded areas indicate the standard deviation. Correlations are computed for the ungrouped data points. While the standard deviation appears moderately high, note that the found trends are consistent over many bins of various layer types. **(B) Wider Layers are More Interpretable.** Average MIS per layer as a function of the relative width of the layer compared to all layers of the same type in the network, grouped by layer types. The values are grouped into 5 bins.

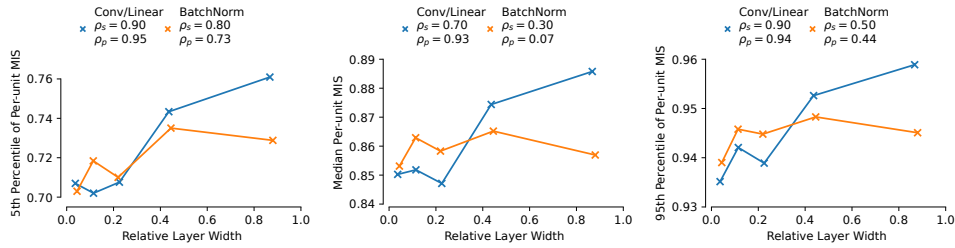

Fig. 24: **Wider Layers are More Interpretable.** Wider layers within a network are moderately more interpretable based on the computed MIS. This trend holds for both the per-layer-average (see Fig. 6B) as well as the 5th percentile, median, or 95th percentile of the per-layer distribution as shown here from left to right. This suggests that the overall distribution is shifted to higher MIS values for wider layers, compared to just a few outliers that positively influence the average value.

Tab. 24: **Pareto-optimal Models for Optimizing ImageNet Accuracy and MIS.** As Fig. 4A shows an anticorrelation between ImageNet top-1 accuracy and MIS, we here list the Pareto-optimal models for optimizing both accuracy and MIS at the same time.

| Model | ImageNet top-1 Accuracy [%] | MIS |
|---|---|---|
| GoogLeNet | 69.15 | 0.908 |
| timm:resnet34.a3_in1k | 72.97 | 0.904 |
| timm:resnet50_gn.a1h_in1k | 81.22 | 0.901 |
| timm:ecaresnet101d_pruned.miil_in1k | 82.00 | 0.985 |
| timm:eva02_small_patch14_336.mim_in22k_ft_in1k | 85.72 | 0.890 |
| timm:vit_base_patch8_224.augreg_in21k_ft_in1k | 85.8 | 0.871 |
| timm:caformer_b36.sail_in1k_384 | 86.41 | 0.870 |
| timm:caformer_s36.sail_in22k_ft_in1k_384 | 86.86 | 0.870 |
| timm:caformer_b36.sail_in22k_ft_in1k_384 | 88.06 | 0.864 |
| timm:beitv2_large_patch16_224.in1k_ft_in22k_in1k | 88.39 | 0.39 |

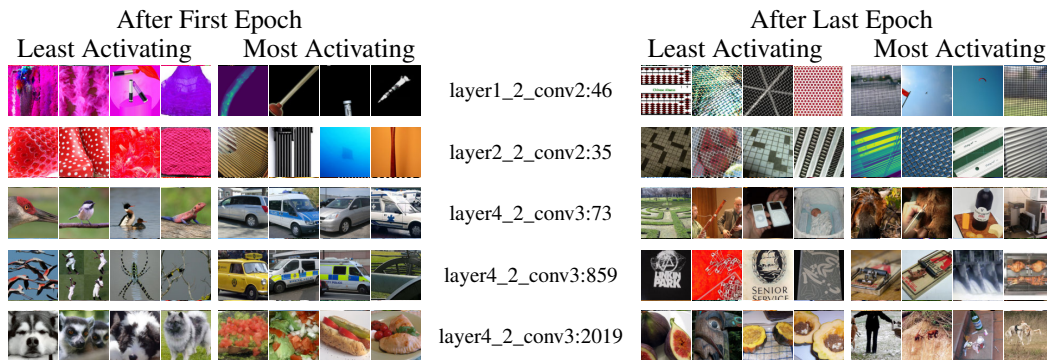

Fig. 25: **How do Dataset Exemplars for Units with Strong MIS Drop Change?** To gain a better understanding of why the MIS of a ResNet50 drops during training after the first epoch, we display the least/most activating dataset exemplars of four units from the model after the first (left) and after the last (right) epoch. While the explanations after the first epoch seem to focus on easy-to-grasp visual features, the units on the right react to less clear-cut concepts. The units are among the units with the strongest MIS drop in the convolutional layers with the strongest MIS drop.

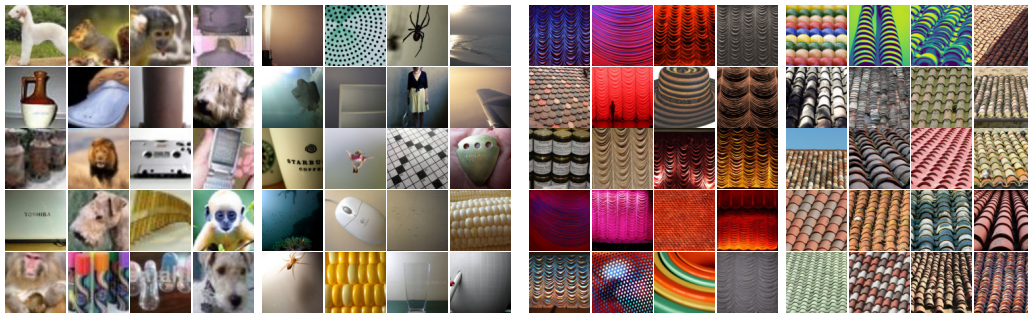

(a) Clip ViT-B32, resblocks_3_mlp_c_proj, unit 573

(b) DenseNet201, block3_layer10_norm1, unit 138

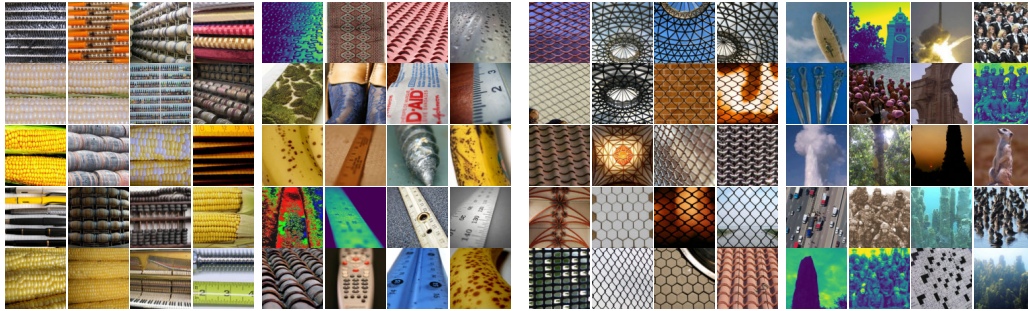

(c) DenseNet201, block3_layer29_conv1, unit 39

(d) DenseNet201, block3_layer35_norm2, unit 123

Fig. 26: **Visualization of Units for which MIS overestimates HIS.** To showcase the shortcomings of the MIS, we visualize four units for which the MIS predicts an interpretability that is higher than the measured HIS in Fig. 2B. See Fig. 27 for the opposite direction. For each unit, we show the 20 most (right) and 20 least (left) activating dataset exemplars.

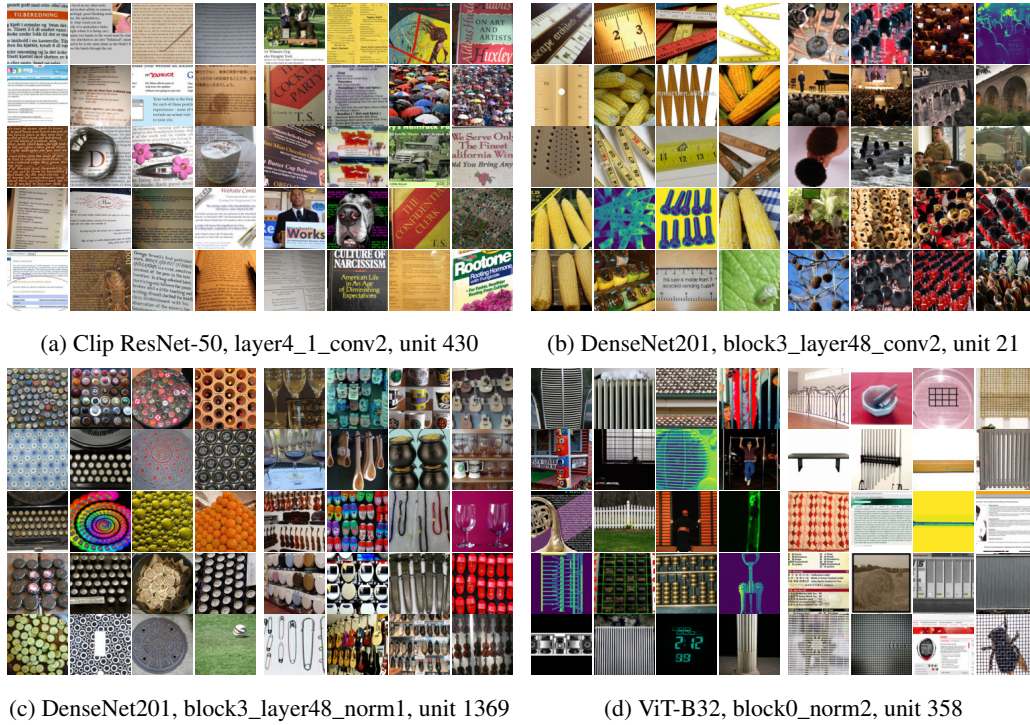

(a) Clip ResNet-50, layer4_1_conv2, unit 430

(b) DenseNet201, block3_layer48_conv2, unit 21

(c) DenseNet201, block3_layer48_norm1, unit 1369

(d) ViT-B32, block0_norm2, unit 358

Fig. 27: **Visualization of Units for which MIS underestimates HIS.** To showcase the shortcomings of the MIS, we visualize four units for which the MIS predicts an interpretability that is lower than the measured HIS in Fig. 2B. See Fig. 26 for the opposite direction. For each unit, we show the 20 most (right) and 20 least (left) activating dataset exemplars.

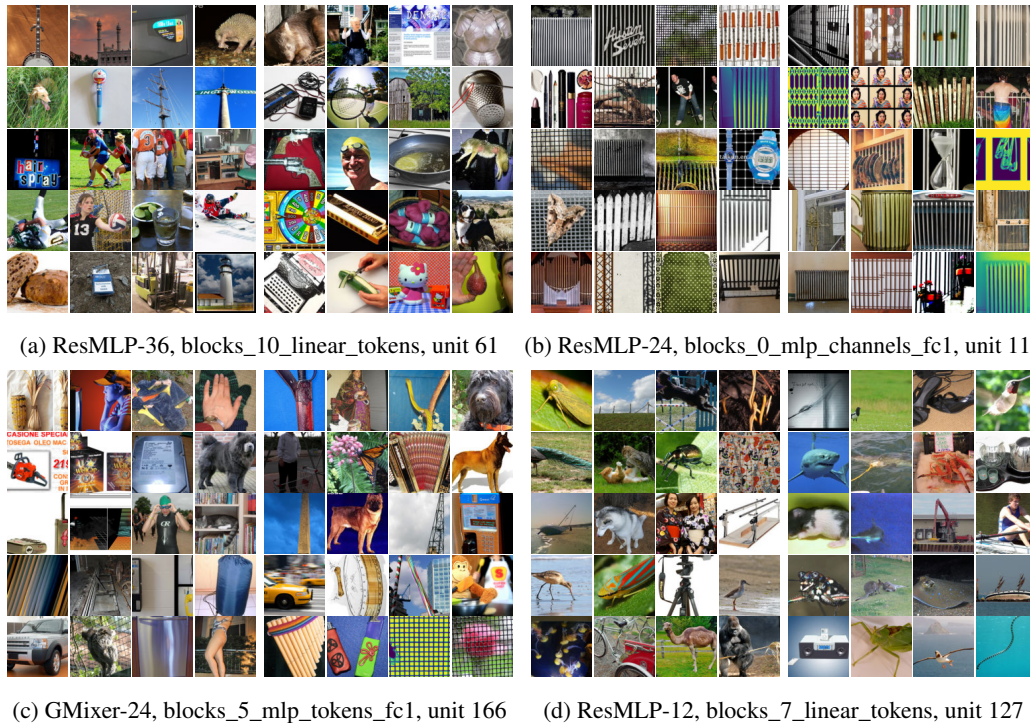

(a) ResMLP-36, blocks_10_linear_tokens, unit 61

(b) ResMLP-24, blocks_0_mlp_channels_fc1, unit 110

(c) GMixer-24, blocks_5_mlp_tokens_fc1, unit 166

(d) ResMLP-12, blocks_7_linear_tokens, unit 127

Fig. 28: **Visualization of Hard Units from Models with High Variability.** For the four models with the highest variability in MIS (see Fig. 4B), we visualize one of the units with the lowest MIS each. For each unit, we show the 20 most (right) and 20 least (left) activating dataset exemplars.

